# Empowering and Assessing the Utility of Large Language Models in Crop Science

**Hang Zhang**[1][*]  **Jiawei Sun**[1][*]  **Renqi Chen**[1][*]  **Wei Liu**[1]  **Zhonghang Yuan**[1]
**Xinzhe Zheng**[1]  **Zhefan Wang**[1]  **Zhiyuan Yang**[4]  **Hang Yan**[1]  **Hansen Zhong**[1]
**Xiqing Wang**[3]  **Wanli Ouyang**[1]  **Fan Yang**[2][†]  **Nanqing Dong**[1][†]
[1]Shanghai Artificial Intelligence Laboratory  [2]Yazhouwan National Laboratory
[3]China Agricultural University  [4]Hangzhou Dianzi University

## Abstract

Large language models (LLMs) have demonstrated remarkable efficacy across knowledge-intensive tasks. Nevertheless, their untapped potential in crop science presents an opportunity for advancement. To narrow this gap, we introduce CROP[3], which includes a novel instruction tuning dataset specifically designed to enhance LLMs' professional capabilities in the crop science sector, along with a benchmark that serves as a comprehensive evaluation of LLMs' understanding of the domain knowledge. The CROP dataset is curated through a task-oriented and LLM-human integrated pipeline, comprising 210,038 single-turn and 1,871 multi-turn dialogues related to crop science scenarios. The CROP benchmark includes 5,045 multiple-choice questions covering three difficulty levels. Our experiments based on the CROP benchmark demonstrate notable enhancements in crop science-related tasks when LLMs are fine-tuned with the CROP dataset. To the best of our knowledge, CROP dataset is the first-ever instruction tuning dataset in the crop science domain. We anticipate that CROP will accelerate the adoption of LLMs in the domain of crop science, ultimately contributing to global food production.

## 1  Introduction

Crop cultivation has posed a consistent and enduring challenge throughout human history. Nevertheless, the ability to guarantee an ample harvest remains uncertain due to various factors, such as weather conditions, regional disparities, and the prevalence of pests and diseases [59, 64]. Without prompt measures, these factors can result in reduced agricultural output, food shortages, and even widespread hunger, particularly in underdeveloped nations [24]. Consequently, it is essential to leverage technological advancements for grain practitioners to enhance production to combat hunger and achieve global food security [25, 26].

Recently, the field of artificial intelligence and natural language processing (NLP) has witnessed rapid advancements in large language models (LLMs) [7, 10, 8]. With instructions, LLMs can generate contexts with professional knowledge regarding users' enquires [34], leading to notable achievements and practical applications in various sectors, including legal consulting [18, 23], clinical management [58, 40], *etc*. In the realm of crop science, LLMs have been investigated in addressing agricultural-related exam questions [55], rendering the opportunity to harness LLMs to improve grain production. Despite the overall acceptable outcomes, LLMs struggle to perform well in specific crop science scenarios, such as pest management [66]. In addition, the agriculture-related exam datasets

---

[*]These authors contributed equally to this work.

[†]Corresponding authors.

[3]The dataset and code are available at website `https://open-sciencelab.github.io/Crop/`.

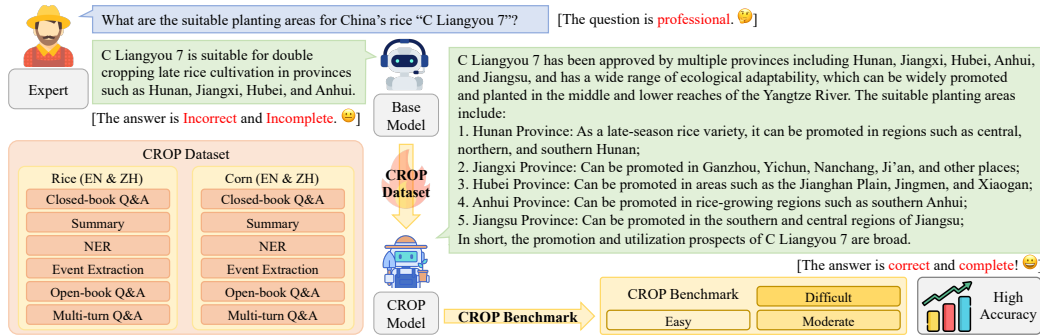

Figure 1: Schematic overview of an intended use case. By fine-tuning a base LLM using the proposed CROP dataset, we obtain a new version whose answer becomes more accurate and comprehensive, which is validated by the proposed CROP benchmark objectively.

used for evaluation, including CCA [2], Embrapa [3], and Agri Exam [1], are not comprehensive in terms of both quantity and locality. Consequently, LLMs are not ready to serve as practical crop science assistants yet.

To harness the full potential of LLMs for crop science, we propose a suite called CROP, which encompasses 1) *an extensive instruction tuning dataset, designed to enhance the domain-specific proficiency of LLMs in crop science.* 2) *a meticulously constructed benchmark, aimed at assessing the performance of LLMs across a variety of domain-related tasks.*

For the CROP dataset, we chose two primary crops, rice, and corn, as they are vital in maintaining worldwide food safety [43, 29]. We collect 290 professional books and 62K research papers in English (EN) and Chinese (ZH), covering six continents on Earth. These materials result in over 210K single-turn and multi-turn dialogues on various scenarios of crop science, such as crop variety selection, resource management, pest control, *etc*. The CROP benchmark includes more than 5K bilingual multiple choice questions (MCQs) in three difficulty levels, derived from over 2K extra academic papers relevant to crop science research. As depicted in Figure 1, utilizing CROP dataset enables LLMs to produce more professional responses to the queries of crop practitioners.

Our core contributions can be summarized as follows:

**An instruction tuning dataset on crop science.** We construct a pioneering, domain-specific instructional tuning dataset tailored for crop science. CROP dataset provides a rich, context-specific resource for training LLMs, leading to more accurate and relevant responses for crop farming issues.

**A benchmark on crop science.** We develop a new benchmark to evaluate LLMs' performance in crop-related tasks. CROP benchmark exceeds prior benchmarks in both quantity and regional coverage, facilitating a more thorough evaluation of LLMs' proficiency in crop science.

**Comprehensive experiments and analysis.** Through extensive experiments, we observe an average accuracy improvement of 29% in selected LLMs after fine-tuning with the CROP dataset, demonstrating the effectiveness of our proposed dataset.

## 2 Dataset Collection

For the CROP dataset, we intend to propose a highly automated data generation pipeline targeting knowledge-intensive scenarios in crop farming practices. A knowledge-intensive scenario is defined as a situation or process that necessitates a substantial amount of specialized knowledge, expertise, and comprehension [42]. Managing microclimates in protected cultivation or understanding pest life cycles for crop protection exemplify such cases in crop science. Dataset generation is LLM-centric with concentrated human intervention incorporated at crucial junctures. If not otherwise indicated, the LLM utilized in this paper is fixed to be GPT-4 [7]. The whole data generation process, as depicted in Figure 3, is a series of transformations that converts raw digital documents to the final dialogue collection. We create a universal mechanism for dialogue task design and develop separate pipelines for the generation of single-turn and multi-turn dialogues.

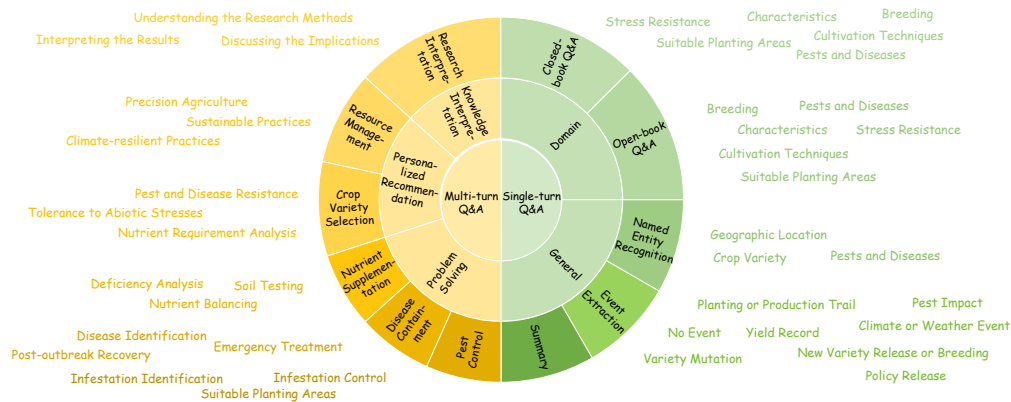

Figure 2: Hierarchical view of tasks in CROP dataset. Dialogues can be single-turn or multi-turn (first tier). The second tier specifies task types. The third tier further decomposes these types into finer-grained tasks. Task-specified topics are rendered around the taxonomy.

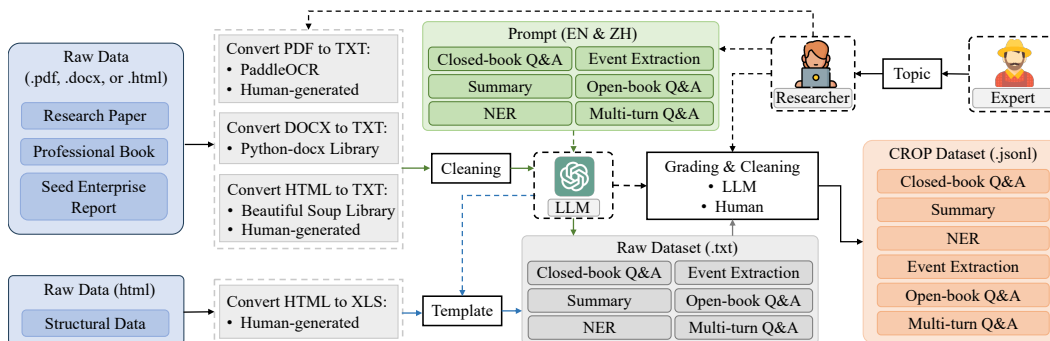

Figure 3: Schematic overview of the dialogue collection procedure. Raw data is first converted to TXT or XLS format using text extraction tools. We then prompt an LLM to either directly generate Q&As from unstructured data or design templates that further transform structured data into dialogue format. After additional filtering steps with both human and LLM involved, we get the CROP dataset. Solid lines represent input/output, and dashed lines indicate operation.

## 2.1 Data Source

We strive to ensure that our raw data originates from a wide range of sources, covering most knowledge-intensive application scenarios in the crop science domain. The raw data in the unstructured format, encompasses research papers, professional books, and corporate reports. In addition, we include human-generated structured raw data, which describes various information about different types of cereals. More details are shown in Section 3.2.

## 2.2 Task Design

We present a framework that synthesizes the expertise of researchers and domain specialists to enhance the task design process. For single-turn dialogues, task types can be either general or domain-specific, with each type compromising specified tasks. For multi-turn dialogues, tasks are defined through a consecutive process that first identifies recommended scenarios and subsequently determines concrete tasks. Tasks in both single-turn and multiple-turn dialogues are designed with an emphasis on practical applicability. Task composition is shown in Figure 2. An in-depth explanation of the task design procedure is shown in Appendix J.1, with examples and mathematical formulations.

## 2.3 Single-turn Q&A

The raw data comes in a variety of structured and unstructured formats. For unstructured raw data, as shown in Figure 3, we choose appropriate text extraction tools for the conversion of document format as the initial step. The information extracted, saved in TXT format, was further refined, ensuring the preservation of specialized knowledge and standardization of the text format. LLM is then utilized to construct single-turn dialogues for a broad spectrum of tasks, using the cleaned texts and task-dependent prompts. For structured raw data, we employ templates generated by the LLM to convert information stored in XLS files into the dialogue format.

For unstructured data, the prompt for the LLM, regardless of tasks, comprises four components: Role Description, Reference Text, Guidelines, and Example Dialogue. This structure adheres to the "Persona-Task-Context-Format" rule as suggested by "Prompting Guide 101" [30]. Role Description employs role-play prompting [36], a widely used approach for enhancing prompt effectiveness. We further refine the method by making the assigned role context-adaptive. Reference Text contains the context depending upon which Q&A pairs are generated. Guidelines consist of detailed, task-dependent restrictions. In addition, we incorporate XML tags and a one-shot demonstration example, applying techniques from "Anthropic User Guides" [11]. We anticipate all generated pairs to exhibit deterministic bias and be strictly constrained in patterns. Examples of prompt-generation pairs are shown in Appendix J.2.

## 2.4 Multi-turn Q&A

In CAMEL [38], a framework for conversation generation for tasks in the problem-solving scenario is proposed. Considering the practice-oriented nature of our intended use cases, we propose a domain-adapted version, as depicted in Figure 4. Domain experts first suggest common application scenarios. For each scenario, an LLM then generates several tasks and assigns a role pair, consisting of an assistant role and a user role, for each task. Dialogues are then generated in a multi-agent setting where the assistant role and the user role are assigned to a user agent and an assistant agent correspondingly to complete a designated task. The prompts for both agents (LLMs) are shown in Appendix J.3. Given the knowledge-intensive nature of covered tasks, we use Retrieval-Augmented Generation (RAG) to enhance the performance of both agents [69].

Given the role pair and task, an assistant agent $A$ and a user agent $U$ engage in a dialogue to collaboratively achieve a goal. The user poses queries, and the assistant is expected to provide responses that address these queries. We denote the query from the user at time $t$ as $q_t$ and the response from the assistant as $r_t$. The collection of dialogue messages up to time $t$ can be represented as $d_t = \{(q_0, r_0), \cdots, (q_t, r_t)\}$.

Specifically, the first query $q_0$ is sampled from a probability distribution over queries $Q_0$, given a context $c_\theta$ that RAG retrieved based on task $\theta$:

$$Q_0 \sim P_U(q_0|c_\theta) \tag{1}$$

At the time step $t + 1$, $U$ takes the historical dialogue message set $d_t$ and an RAG-retrieved context $c_{d_t}$ to provide a new query $q_{t+1}$, sampled from a probability distribution over queries $Q_{t+1}$:

$$Q_{t+1} \sim P_U(q_{t+1}|d_t, c_{d_t}) \tag{2}$$

The new query $Q_{t+1}$, along with the dialogue set $d_t$ and RAG-retrieved context $c_{d_t \cup \{q_{t+1}\}}$, is then passed to $A$, who responds with a message $r_{t+1}$, sampled from a probability distribution over responses $R_{t+1}$:

$$R_{t+1} \sim P_A(r_{t+1}|d_t, q_{t+1}, c_{d_t \cup \{q_{t+1}\}}). \tag{3}$$

After obtaining the response $r_{t+1}$ to the query $q_{t+1}$, the dialogue set is updated:

$$d_{t+1} = d_t \cup \{(q_{t+1}, r_{t+1})\}. \tag{4}$$

The above formulation models a dynamic and interactive dialogue that evolves over time. While this novel framework introduces RAG-retrieved content to refine generations to be context-dependent, this additional content can be highly customized depending on specific use cases. This new mechanism enhances interpretability, providing fresh insights for building a more predictable and controllable generation system.

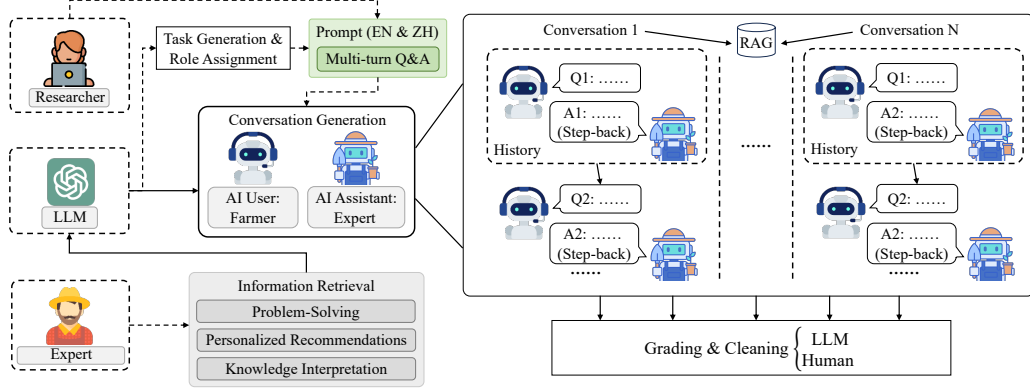

Figure 4: Schematic overview of the multi-turn dialogue dataset collection procedure. An LLM creates tasks under the guidance of domain experts and assigns roles to two agents. Using task-dependent prompts from researchers, the LLM generates dialogues with RAG. Additional filtering steps are then conducted. Solid lines represent input/output, while dashed lines indicate operation.

## 2.5 Filtering

To ensure the quality of generated Q&A pairs, we conduct two stages of evaluation and filtering, with both LLMs and human domain experts involved.

In the first stage, an LLM functions as a grader, quantifying the quality of each generated pair based on pre-defined metrics. For single-turn Q&A, we focus on correctness, professionalism, and richness. For multi-turn Q&A, we prioritize coherence, consistency, and specificity. For each task, let $P$ be the set of all generated Q&A pairs, with $|P| = N$. We first create two permutations $D_1$ and $D_2$ of $P$, each containing $N$ pairs in a random order. Each permutation $D_i$ is further split into groups of 4 pairs in order:

$$D_i = \{G_{i1}, G_{i2}, \ldots, G_{im}\}, m = \left\lceil \frac{N}{4} \right\rceil. \quad (5)$$

For each group $G_{ij}$, an LLM ranks the pairs based on the quality metrics and retains the top two, resulting in $\text{Top}_{ij} = \text{SelectTop2}(G_{ij})$. Let $R_1$ and $R_2$ be the sets of retained pairs after ranking in $D_1$ and $D_2$ respectively. The final set of selected pairs $R$ are those retained in both permutations:

$$R_1 = \bigcup_{j=1}^{m} \text{Top}_{1j}, R_2 = \bigcup_{j=1}^{m} \text{Top}_{2j}, R = R_1 \cap R_2. \quad (6)$$

The second stage of evaluation and filtering involves only human experts, further enhancing the quality of the dataset. More details are shown in Appendix F.

## 3 Dataset Analysis

Our CROP dataset focuses on rice and corn, comprising single-turn and multi-turn dialogues.

### 3.1 Basic Statistics

The single-turn dialogues comprise 210,038 high-quality samples, while the multi-turn dialogues include 1,871 high-quality samples. Each task within the multi-turn dialogues possesses a minimum of 80 samples, showcasing the dataset's superiority. In terms of grain species, the CROP dataset contains 140,056 dialogue samples for rice and 69,482 for corn. For more detailed compositions, please refer to Appendix Tables 4 and 5.

### 3.2 Diversity

We demonstrate that our dataset is diverse in terms of:

**Data Source.** The data source distribution is diverse for both rice and corn. For the rice part, we use 40,078 research papers from the Web of Science [6], 186 professional books [4], 645 crop enterprise reports, and human-generated structural data of 5,990 rice species, containing their characteristic information such as rice blast grade and suitable cultivation area. For the corn part, we use 22,056 research papers, 104 professional books, and 1,841 crop enterprise reports.

**Geographic Diversity.** In our dataset, the sources and cultivation areas of different types of grains originate from various regions of the world. For example, we record Japan if an article introduces a certain grain that was first cultivated in Japan; we record China if an article discusses whether a particular region in China is suitable for grain cultivation. For rice and corn, our data involves 33 and 43 countries across six continents respectively. All geographical regions involved are shown in Appendix Table 6 and 7.

**Content Length.** We conduct a statistical analysis of the content length measured by token counts (using the official GPT-4 tokenizer) regarding grain species, languages, and tasks in the single-turn dialogue dataset. The results are illustrated in Appendix Table 8. Of all the tasks, closed-book Q&A (CQA) has the shortest average length at 128 tokens, while the summary task has the longest average length at 446 tokens. The average lengths of rice and corn samples show no significant variation. As for language perspective, the Chinese samples are considerably longer, approximately twice the length of the English samples. In the multi-turn dialogue dataset, most samples are of 4 rounds (85%), with a small portion (15%) of 3 or 5 rounds, shown in Appendix Figure 8.

**Content Distribution.** We focus on the content distribution of the domain Q&A set (including closed-book and open-book Q&A) which mainly consists of professional knowledge and then found that it encompasses a diverse range of topics. For Chinese Q&A samples, topics cover maize (corn) breeding, hybrid rice, growth characteristics, *etc*. For English Q&A samples, topics include selection, resistance, maize (corn), *etc*. More details are provided in Appendix Figure 9 and 10, which illustrate the information cartography of the Chinese and English Q&A samples respectively.

# 4 Benchmark

To quantitatively evaluate the performance of both commercial LLMs and cutting-edge open-source LLMs fine-tuned with the CROP dataset, we present the CROP benchmark. To avoid disputes arising from subjective questions, our benchmark consists of only MCQs, with a label of difficulty level (easy, moderate, difficult) assigned to each.

## 4.1 Data Source

Additionally, we select 4,018 literature articles in both Chinese and English, on rice and corn for the generation of benchmarks, distinct from the literature used to construct the CROP dataset. To ensure the data source quality, all literature articles are downloaded from the Web of Science [6].

## 4.2 Pipeline Design

Q&A pairs, utilized for benchmarking objectives, are not generated in the same manner as CROP dataset that employs a task-oriented pipeline. We abstain from incorporating directives from researchers and domain experts, or from defining specific tasks beforehand, instead allowing the model to independently determine the actual format and content of the Q&A pair. By imposing fewer constraints, we intend to furnish a benchmark that facilitates a comprehensive evaluation of an LLM's capabilities for domain-knowledge tasks, thereby benefiting the broader community.

Using the same pre-processing method as the dataset, we convert the initial PDF files into TXT format and feed them into GPT-4 to generate Q&A pairs. Specifically, our prompts are tailored to closed-book MCQ generation because we prioritize a model's knowledge retrieval capability. The prompt for generating the benchmark dataset is provided in Appendix J.4, The content of Q&A pairs is related to the knowledge of crop science. Specifically, to ensure the fairness and quality of the

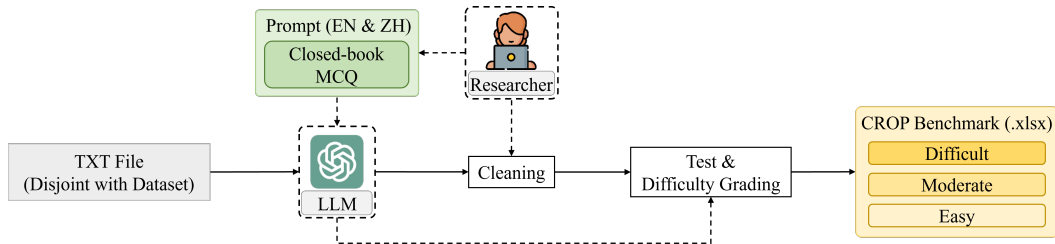

Figure 5: Benchmark pipeline overview. We prompt an LLM to generate MCQs from TXT files. After additional filtering steps with both human and LLM involved, we get the CROP benchmark, comprising three difficulty levels.

benchmark, we impose restrictions on their generation, including equal lengths and a certain level of differentiation in content. From a corpus of over 2,000 articles, we extract over 6,000 MCQs. To ensure the quality of MCQs, we conduct manual screening and select 5,045 MCQs as the final benchmark, which are subsequently categorized according to difficulty levels.

## 4.3 Difficulty Specification

We classify the 5,045 questions in the benchmark into three difficulty levels: easy, moderate, and difficult, using GPT-4 and GPT-3.5. Easy questions are those both models answered correctly, moderate questions are those answered correctly only by GPT-4, and difficult questions are those answered incorrectly by GPT-4. The statistics are shown in Table 1.

Table 1: Proportion of Difficulty.

| Level | Count | Proportion |
|---|---|---|
| Easy | 1,613 | 31.97% |
| Moderate | 2,754 | 53.72% |
| Difficult | 722 | 14.31% |

## 4.4 Topic Distribution

We conduct a statistical analysis of the content distribution in the CROP benchmark. For corn-related MCQs, the most frequently occurring topics are corn variety, biological characteristics, and genetics of corn. For rice-related MCQs, the most frequently occurring topics are hybrid rice, rice characteristics, and pests and diseases. We represent the main topics of our benchmark in Appendix Figure 11, which almost cover topics that are currently of concern to experts in the crop science domain. This illustrates the practical value of the CROP benchmark.

## 4.5 Benchmark Comparison

We compare our benchmark with other evaluation benchmarks in the crop science domain. CROP benchmark consists of 5045 Chinese and English MCQs and covers 22 countries across six continents, surpassing existing agriculture-related question databases[1, 2, 3] in terms of language types, size, and geographic coverage. Details are presented in Appendix I.2.

# 5 Experiments

We employ the CROP benchmark to evaluate the performance of currently accessible LLMs and conduct a comprehensive analysis.

## 5.1 Model Selection

The LLMs we benchmarked can be divided into two groups: 1) *Commercial LLMs*: this group comprises GPT-3.5 [15], GPT-4 [7], Claude-3 [10], and Qwen [12], all accessible via their API service. 2) *Open-source LLMs*: this group contains LLaMA3-8B [8], Qwen1.5-7B [12], and InternLM2-7B [16], as well as their fine-tuned versions on CROP dataset. We provide training details of open-source LLMs in Appendix K.

Table 2: Performance of selected LLMs on the CROP benchmark. We indicate the accuracy changes of the fine-tuned LLMs compared to the base models in blue, where the accuracy has generally improved when the CROP dataset is used in fine-tuning.

| Model | Access | Size | Overall ↑ | Difficulty | | |
|---|---|---|---|---|---|---|
| | | | | Easy ↑ | Moderate ↑ | Difficult ↑ |
| *Commercial LLMs* | | | | | | |
| GPT-4[1] | API | N/A | 0.856 | 1.000[2] | 1.000[2] | 0.000[2] |
| GPT-3.5[1] | API | N/A | 0.328 | 1.000[2] | 0.000[2] | 0.061 |
| Claude-3[1] | API | N/A | 0.900 | 0.982 | 0.968 | 0.458 |
| Qwen[1] | API | N/A | 0.866 | 0.987 | 0.945 | 0.301 |
| *Open-source LLMs* | | | | | | |
| LLaMA3-Base | Weights | 8B | 0.348 | 0.443 | 0.341 | 0.161 |
| +CQIA | Weights | 8B | 0.643 (+0.295) | 0.791 (+0.348) | 0.651 (+0.310) | 0.281 (+0.120) |
| +CROP | Weights | 8B | 0.752 (+0.404) | 0.866 (+0.432) | 0.772 (+0.431 | 0.378 (+0.217) |
| +CQIA+CROP | Weights | 8B | 0.754 (+0.406) | 0.918 (+0.475) | 0.779 (+0.438) | 0.295 (+0.134) |
| Qwen1.5-Base | Weights | 7B | 0.646 | 0.799 | 0.646 | 0.302 |
| +CQIA | Weights | 7B | 0.688 (+0.042) | 0.880 (+0.081) | 0.689 (+0.043) | 0.258 (-0.044) |
| +CROP | Weights | 7B | 0.676 (+0.030) | 0.849 (+0.050) | 0.688 (+0.042) | 0.202 (-0.100) |
| +CQIA+CROP | Weights | 7B | 0.709 (+0.063) | 0.910 (+0.111) | 0.704 (+0.058) | 0.227 (-0.075) |
| InternLM2-Base | Weights | 7B | 0.368 | 0.445 | 0.381 | 0.148 |
| +CQIA | Weights | 7B | 0.723 (+0.355) | 0.861 (+0.416) | 0.750 (+0.369) | 0.317 (+0.169) |
| +CROP | Weights | 7B | 0.748 (+0.380) | 0.945 (+0.500) | 0.761 (+0.380) | 0.212 (+0.064) |
| +CQIA+CROP | Weights | 7B | 0.768 (+0.400) | 0.939 (+0.494) | 0.794 (+0.413) | 0.285 (+0.137) |

[1] GPT-4 API is "gpt-4-turbo-2024-04-09". GPT-3.5 API is "gpt-3.5-turbo-0125".
Claude-3 API is "claude-3-opus-20240229". Qwen API is "qwen-max".
[2] The accuracy of 0.000 and 1.000 is explained in Section 4.3.

## 5.2 Evaluation Protocol

We assess model performance on CROP benchmark using accuracy. Owing to our adoption of text generation for eliciting responses from various LLMs, we annotate answer options accompanying each question with Roman numerals (I, II, III, IV) when designing prompts. To match the generated responses with the answer options, regular expressions are utilized to identify and extract the Roman numerals. In cases where this approach fails to accurately decipher the response, we resort to manual inspection by domain experts, thereby ensuring accurate calculation of accuracy.

## 5.3 Performance Comparison

Table 2 showcases the performance of selected LLMs on the CROP benchmark. It examines whether commercial LLMs possess professional knowledge in crop science and if the capabilities of open-source LLMs can be enhanced when fine-tuned with the CROP dataset. For commercial models, Claude-3 achieves the highest overall accuracy (0.900), while GPT-3.5 lags behind (0.328). Even though GPT-4, Claude-3, and Qwen show acceptable general performance, they struggle with difficult tasks, demonstrating the rationality of difficulty level division and the efficacy of the CROP benchmark. In the open-source category, we choose CQIA [5] as a general-task instruction tuning dataset to enhance the limited instruction-following capability of the base models [68]. Building on this, we employ the CROP dataset for additional training to confirm that it can enhance an LLM's performance in responding to crop science-related queries. The findings indicate that when further fine-tuned with the CROP dataset, there is an average improvement of 9.2%. Moreover, we can observe that in most cases, the improvements brought by CROP are relatively more compared with CQIA. The comprehensive results underscore the efficacy of CROP dataset in fostering the professional capabilities of LLMs.

## 5.4 Analysis on Training Epochs

To evaluate the performance of fine-tuned LLMs under different training epochs, we conduct an experiment in terms of training epochs, with the results summarized in Table 3. Different open-source LLMs show distinct convergence tendencies. For instance, LLaMA3 reaches optimal performance

Table 3: Performance of LLMs trained under different epochs. The highest value for each model on different tasks is **bolded**. Variation denotes the accuracy difference between Chinese and English. More experiments on training epochs when the model is fine-tuned only on the CROP dataset are elaborated in Appendix 14.

| Model | Epoch | Size | Overall ↑ | Difficulty | | | Language | | |
|---|---|---|---|---|---|---|---|---|---|
| | | | | Easy ↑ | Moderate ↑ | Difficult ↑ | Chinese ↑ | English ↑ | Variation ↓ |
| LLaMA3-Base | N/A | 8B | 0.348 | 0.443 | 0.341 | 0.161 | 0.327 | 0.369 | 4.2% |
| +CQIA+CROP | 1 | 8B | 0.738 | 0.903 | 0.758 | 0.292 | 0.719 | 0.757 | 3.8% |
| +CQIA+CROP | 2 | 8B | 0.742 | 0.902 | 0.772 | 0.271 | 0.729 | 0.755 | **2.6%** |
| +CQIA+CROP | 4 | 8B | **0.754** | **0.918** | **0.779** | **0.295** | **0.738** | **0.770** | 3.2% |
| Qwen1.5-Base | N/A | 7B | 0.646 | 0.799 | 0.646 | **0.302** | 0.667 | 0.624 | 4.3% |
| +CQIA+CROP | 1 | 7B | 0.702 | **0.910** | **0.717** | 0.183 | **0.725** | 0.680 | 4.5% |
| +CQIA+CROP | 2 | 7B | 0.670 | 0.875 | 0.677 | 0.181 | 0.690 | 0.649 | 4.1% |
| +CQIA+CROP | 4 | 7B | **0.709** | **0.910** | 0.704 | 0.227 | 0.717 | **0.686** | **3.1%** |
| InternLM2-Base | N/A | 7B | 0.368 | 0.445 | 0.381 | 0.148 | 0.409 | 0.327 | 8.2% |
| +CQIA+CROP | 1 | 7B | 0.764 | **0.942** | 0.787 | 0.276 | 0.770 | 0.757 | 3.3% |
| +CQIA+CROP | 2 | 7B | **0.809** | 0.909 | **0.855** | **0.414** | **0.811** | **0.807** | **0.4%** |
| +CQIA+CROP | 4 | 7B | 0.768 | 0.939 | 0.794 | 0.285 | 0.770 | 0.766 | **0.4%** |

following four epochs, whereas InternLM2 attains peak performance after just two epochs of training. Contrarily, the convergence trend of Qwen1.5 appears to be relatively unstable, and its performance enhancement (6.3%) is marginal compared to its base version. More experiments on training epochs when the model is fine-tuned only on the CROP dataset are elaborated in Appendix 14.

## 5.5 Analysis on Languages

Given that our benchmark includes both Chinese and English, the performance of LLMs on questions in these two languages has been reported in Table 3. After four epochs of training with the CROP dataset, models did not exhibit a remarkable language bias, as variations of accuracy across various languages fall within the range of 0.4% to 3.2%. These results underscore the robustness of the model in multilingual contexts, ensuring its applicability in diverse linguistic scenarios.

# 6 Limitations

The construction of the CROP dataset faces several limitations. For raw data, its collection focuses primarily on a selected number of crop species, specifically corn and rice. Other prevalent crop families, such as wheat, tuber crops, and beans, were rarely included. The format of the source data is text-only. While this choice ensures compatibility with the GPT-4 API, the omission of data in other modalities may limit the richness of the dataset. Crop science is a dynamically evolving field. New findings and research studies should be integrated to keep the dataset up-to-date. In terms of the data construction pipeline, extraction tools used during data collection introduce errors, while efforts to enhance their performance are still ongoing. LLMs, when generating Q&A pairs, may exhibit biases due to their training data. Task design influenced by researchers and domain experts can also introduce bias towards certain question types. In the experiment aspect, rigorous empirical investigations are necessitated to ascertain the efficacy of the suggested dataset in fine-tuning LLMs with other parameter magnitudes (70B, 110B). Detailed discussion is shown in Appendix D.

# 7 Conclusion

In this paper, we propose the CROP dataset to improve the professional capabilities of LLMs in the crop science domain. Additionally, to compensate for the absence of an open-source benchmark for evaluating models' expertise in this domain, we introduce the CROP benchmark, which comprises MCQs for objective assessment. We hope that the proposed dataset and benchmark can foster AI research in crop science, facilitate knowledge transfer for agricultural practitioners, enhance crop yields, and contribute to solving hunger issues.

## 8 Acknowledgements

This work was supported by Shanghai Artificial Intelligence Laboratory, the Yazhouwan National Laboratory Project (grant no. 2310CF01), and the Hainan Yazhou Bay Seed Laboratory Project (grant no. B21HJ0001). We thank YZBSTCACC for data storage support.

## Footnotes

[4]Professional books, crop enterprise reports, and human-generated structural data are provided by Yazhouwan National Laboratory and China Agricultural University.

[5]https://neurips.cc/public/EthicsGuidelines

## References

[1] Agri exam. agriculture previous year question paper. https://www.agriexam.com/agriculture-previous-year-question-paper. Accessed: 2024-05-21.

[2] Certified crop adviser. become certified: Certifications. https://www.certifiedcropadviser.org/become-certified/certifications/. Accessed: 2024-05-21.

[3] Embrapa. mais500p500r. https://mais500p500r.sct.embrapa.br/view/index.php. Accessed: 2024-05-21.

[4] Fruit robot. https://www.raussendorf.de/en/fruit-robot.html.

[5] Coig-cqia: Quality is all you need for chinese instruction fine-tuning. https://github.com/paralym/COIG-CQIA, 2023.

[6] Web of science, 2024. Accessed May 30, 2024.

[7] Josh Achiam, Steven Adler, Sandhini Agarwal, Lama Ahmad, Ilge Akkaya, Florencia Leoni Aleman, Diogo Almeida, Janko Altenschmidt, Sam Altman, Shyamal Anadkat, et al. Gpt-4 technical report. *arXiv preprint arXiv:2303.08774*, 2023.

[8] AI@Meta. Llama 3 model card. 2024.

[9] Olabimpe Banke Akintuyi. Ai in agriculture: A comparative review of developments in the usa and africa. *Research Journal of Science and Engineering*, 10(02):060–070, 2024.

[10] Anthropic. The claude 3 model family: Opus, sonnet, haiku. Technical report, 2024.

[11] Anthropic. Prompt engineering techniques, 2024.

[12] Jinze Bai, Shuai Bai, Yunfei Chu, Zeyu Cui, Kai Dang, Xiaodong Deng, Yang Fan, Wenbin Ge, Yu Han, Fei Huang, et al. Qwen technical report. *arXiv preprint arXiv:2309.16609*, 2023.

[13] Simon Birrell, Josie Hughes, Julia Y Cai, and Fumiya Iida. A field-tested robotic harvesting system for iceberg lettuce. *Journal of Field Robotics*, 37(2):225–245, 2020.

[14] Sebastian Bosse, Eike Gassen, Mortesa Hussaini, and Peter Eisert. Platform-based and ai-enabled farming to foster agricultural ecosystems.

[15] Tom Brown, Benjamin Mann, Nick Ryder, Melanie Subbiah, Jared D Kaplan, Prafulla Dhariwal, Arvind Neelakantan, Pranav Shyam, Girish Sastry, Amanda Askell, et al. Language models are few-shot learners. *Advances in neural information processing systems*, 33:1877–1901, 2020.

[16] Zheng Cai, Maosong Cao, Haojiong Chen, Kai Chen, Keyu Chen, Xin Chen, Xun Chen, Zehui Chen, Zhi Chen, Pei Chu, et al. Internlm2 technical report. *arXiv preprint arXiv:2403.17297*, 2024.

[17] Yupeng Chang, Xu Wang, Jindong Wang, Yuan Wu, Linyi Yang, Kaijie Zhu, Hao Chen, Xiaoyuan Yi, Cunxiang Wang, Yidong Wang, et al. A survey on evaluation of large language models. *ACM Transactions on Intelligent Systems and Technology*, 15(3):1–45, 2024.

[18] Jiaxi Cui, Zongjian Li, Yang Yan, Bohua Chen, and Li Yuan. Chatlaw: Open-source legal large language model with integrated external knowledge bases, 2023.

[19] Nadia Delavarpour, Cengiz Koparan, John Nowatzki, Sreekala Bajwa, and Xin Sun. A technical study on uav characteristics for precision agriculture applications and associated practical challenges. *Remote Sensing*, 13(6):1204, 2021.

[20] Ning Ding, Yulin Chen, Bokai Xu, Yujia Qin, Zhi Zheng, Shengding Hu, Zhiyuan Liu, Maosong Sun, and Bowen Zhou. Enhancing chat language models by scaling high-quality instructional conversations. *arXiv preprint arXiv:2305.14233*, 2023.

[21] Haodong Duan, Jueqi Wei, Chonghua Wang, Hongwei Liu, Yixiao Fang, Songyang Zhang, Dahua Lin, and Kai Chen. Botchat: Evaluating llms' capabilities of having multi-turn dialogues, 2023.

[22] Tom Duckett, Simon Pearson, Simon Blackmore, Bruce Grieve, Wen-Hua Chen, Grzegorz Cielniak, Jason Cleaversmith, Jian Dai, Steve Davis, Charles Fox, et al. Agricultural robotics: the future of robotic agriculture. *arXiv preprint arXiv:1806.06762*, 2018.

[23] Zhiwei Fei, Xiaoyu Shen, Dawei Zhu, Fengzhe Zhou, Zhuo Han, Songyang Zhang, Kai Chen, Zongwen Shen, and Jidong Ge. Lawbench: Benchmarking legal knowledge of large language models. *arXiv preprint arXiv:2309.16289*, 2023.

[24] Food and Agriculture Organization of the United Nations. *The State of Food and Agriculture 2023: Revealing the true cost of food to transform agrifood systems*. FAO, Rome, 2023.

[25] Food and Agriculture Organization of the United Nations (FAO). Leave no one behind: Better production, better nutrition, a better environment and a better life, 2022.

[26] Food and Agriculture Organization of the United Nations (FAO). Goal 2: End hunger, achieve food security and improved nutrition and promote sustainable agriculture, 2024.

[27] J Gai, L Tang, and BL Steward. Automated crop plant detection based on the fusion of color and depth images for robotic weed control. j field robot 37: 35–52, 2020.

[28] Timnit Gebru, Jamie Morgenstern, Briana Vecchione, Jennifer Wortman Vaughan, Hanna Wallach, Hal Daumé Iii, and Kate Crawford. Datasheets for datasets. *Communications of the ACM*, 64(12):86–92, 2021.

[29] H Charles J Godfray and Tara Garnett. Food security and sustainable intensification. *Philosophical transactions of the Royal Society B: biological sciences*, 369(1639):20120273, 2014.

[30] Google. Prompting guide 101. 2024.

[31] Lin Haibo, Dong Shuliang, Liu Zunmin, and Yi Chuijie. Study and experiment on a wheat precision seeding robot. *Journal of Robotics*, 2015:12–12, 2015.

[32] Edward J Hu, Yelong Shen, Phillip Wallis, Zeyuan Allen-Zhu, Yuanzhi Li, Shean Wang, Lu Wang, and Weizhu Chen. Lora: Low-rank adaptation of large language models. *arXiv preprint arXiv:2106.09685*, 2021.

[33] Mohd Javaid, Abid Haleem, Ibrahim Haleem Khan, and Rajiv Suman. Understanding the potential applications of artificial intelligence in agriculture sector. *Advanced Agrochem*, 2(1):15–30, 2023.

[34] Sijie Ji, Xinzhe Zheng, and Chenshu Wu. Hargpt: Are llms zero-shot human activity recognizers? *arXiv preprint arXiv:2403.02727*, 2024.

[35] Nazmuzzaman Khan, Gregory Medlock, Scott Graves, and Sohel Anwar. Gps guided autonomous navigation of a small agricultural robot with automated fertilizing system. Technical report, SAE Technical Paper, 2018.

[36] Aobo Kong, Shiwan Zhao, Hao Chen, Qicheng Li, Yong Qin, Ruiqi Sun, and Xin Zhou. Better zero-shot reasoning with role-play prompting. *arXiv preprint arXiv:2308.07702*, 2023.

[37] Matheus Thomas Kuska, Mirwaes Wahabzada, and Stefan Paulus. Ai for crop production–where can large language models (llms) provide substantial value? *Computers and Electronics in Agriculture*, 221:108924, 2024.

[38] Guohao Li, Hasan Hammoud, Hani Itani, Dmitrii Khizbullin, and Bernard Ghanem. Camel: Communicative agents for" mind" exploration of large language model society. *Advances in Neural Information Processing Systems*, 36, 2024.

[39] Maria Teresa Linaza, Jorge Posada, Jürgen Bund, Peter Eisert, Marco Quartulli, Jürgen Döllner, Alain Pagani, Igor G. Olaizola, Andre Barriguinha, Theocharis Moysiadis, et al. Data-driven artificial intelligence applications for sustainable precision agriculture. *Agronomy*, 11(6):1227, 2021.

[40] Fenglin Liu, Tingting Zhu, Xian Wu, Bang Yang, Chenyu You, Chenyang Wang, Lei Lu, Zhangdaihong Liu, Yefeng Zheng, Xu Sun, et al. A medical multimodal large language model for future pandemics. *NPJ Digital Medicine*, 6(1):226, 2023.

[41] D Longo, A Pennisi, R Bonsignore, G Muscato, and G Schillaci. A multifunctional tracked vehicle able to operate in vineyards using gps and laser range-finder technology. 2010.

[42] Kathryn S McCarthy, Tricia A Guerrero, Kevin M Kent, Laura K Allen, Danielle S McNamara, Szu-Fu Chao, Jonathan Steinberg, Tenaha O'Reilly, and John Sabatini. Comprehension in a scenario-based assessment: Domain and topic-specific background knowledge. *Discourse Processes*, 55(5-6):510–524, 2018.

[43] Sumithra Muthayya, Jonathan D Sugimoto, Scott Montgomery, and Glen F Maberly. An overview of global rice production, supply, trade, and consumption. *Annals of the new york Academy of Sciences*, 1324(1):7–14, 2014.

[44] Luiz FP Oliveira, António P Moreira, and Manuel F Silva. Advances in agriculture robotics: A state-of-the-art review and challenges ahead. *Robotics*, 10(2):52, 2021.

[45] Jiao Ou, Jiayu Wu, Che Liu, Fuzheng Zhang, Di Zhang, and Kun Gai. Inductive-deductive strategy reuse for multi-turn instructional dialogues, 2024.

[46] PaddlePaddle. Paddleocr: Awesome multilingual ocr toolkits based on paddlepaddle. https://github.com/PaddlePaddle/PaddleOCR, 2024.

[47] Dimitris S. Paraforos, Vangelis Vassiliadis, Dietrich Kortenbruck, Kostas Stamkopoulos, Vasileios Ziogas, Athanasios A. Sapounas, and Hans W. Griepentrog. A farm management information system using future internet technologies. *IFAC-PapersOnLine*, 49(16):324–329, 2016. 5th IFAC Conference on Sensing, Control and Automation Technologies for Agriculture AGRICONTROL 2016.

[48] Samyam Rajbhandari, Jeff Rasley, Olatunji Ruwase, and Yuxiong He. Zero: Memory optimizations toward training trillion parameter models. In *SC20: International Conference for High Performance Computing, Networking, Storage and Analysis*, pages 1–16. IEEE, 2020.

[49] Jeff Rasley, Samyam Rajbhandari, Olatunji Ruwase, and Yuxiong He. Deepspeed: System optimizations enable training deep learning models with over 100 billion parameters. In *Proceedings of the 26th ACM SIGKDD International Conference on Knowledge Discovery & Data Mining*, pages 3505–3506, 2020.

[50] Saed Rezayi, Zhengliang Liu, Zihao Wu, Chandra Dhakal, Bao Ge, Chen Zhen, Tianming Liu, and Sheng Li. Agribert: Knowledge-infused agricultural language models for matching food and nutrition. In *IJCAI*, pages 5150–5156, 2022.

[51] Junhu Ruan, Hua Jiang, Chunsheng Zhu, Xiangpei Hu, Yan Shi, Tianjun Liu, Weizhen Rao, and Felix Tung Sun Chan. Agriculture iot: Emerging trends, cooperation networks, and outlook. *IEEE Wireless Communications*, 26(6):56–63, 2019.

[52] Noa Schor, Avital Bechar, Timea Ignat, Aviv Dombrovsky, Yigal Elad, and Sigal Berman. Robotic disease detection in greenhouses: combined detection of powdery mildew and tomato spotted wilt virus. *IEEE Robotics and Automation Letters*, 1(1):354–360, 2016.

[53] Dmitrii Shadrin, Alexander Menshchikov, Andrey Somov, Gerhild Bornemann, Jens Hauslage, and Maxim Fedorov. Enabling precision agriculture through embedded sensing with artificial intelligence. *IEEE Transactions on Instrumentation and Measurement*, 69(7):4103–4113, 2019.

[54] Shivangi Sharma, Kirti Verma, and Palak Hardaha. Implementation of artificial intelligence in agriculture. *Journal of Computational and Cognitive Engineering*, 2(2):155–162, 2023.

[55] Bruno Silva, Leonardo Nunes, Roberto Estevão, and Ranveer Chandra. Gpt-4 as an agronomist assistant? answering agriculture exams using large language models. *arXiv preprint arXiv:2310.06225*, 2023.

[56] Salah Sukkarieh. Mobile on-farm digital technology for smallholder farmers. 2017.

[57] Yuchong Sun, Che Liu, Kun Zhou, Jinwen Huang, Ruihua Song, Wayne Xin Zhao, Fuzheng Zhang, Di Zhang, and Kun Gai. Parrot: Enhancing multi-turn instruction following for large language models, 2024.

[58] Arun James Thirunavukarasu, Darren Shu Jeng Ting, Kabilan Elangovan, Laura Gutierrez, Ting Fang Tan, and Daniel Shu Wei Ting. Large language models in medicine. *Nature medicine*, 29(8):1930–1940, 2023.

[59] Louis M Thompson. Weather variability, climatic change, and grain production: Weather variability is a much more important consideration in grain production than a cooling trend. *Science*, 188(4188):535–541, 1975.

[60] DC Tsouros, S Bibi, and PG Sarigiannidis. A review on uav-based applications for precision agriculture. information 10 (11): 349, 2019.

[61] Parthasarathy Velusamy, Santhosh Rajendran, Rakesh Kumar Mahendran, Salman Naseer, Muhammad Shafiq, and Jin-Ghoo Choi. Unmanned aerial vehicles (uav) in precision agriculture: Applications and challenges. *Energies*, 15(1):217, 2021.

[62] Cor Verdouw, Sjaak Wolfert, George Beers, Harald Sundmaeker, and Grigoris Chatzikostas. Iof2020: Fostering business and software ecosystems for large-scale uptake of iot in food and farming. In *The International Tri-Conference for Precision Agriculture in 2017*. Precision Agriculture Association New Zealand Hamilton (NZ), 2017.

[63] Manas Wakchaure, BK Patle, and AK Mahindrakar. Application of ai techniques and robotics in agriculture: A review. *Artificial Intelligence in the Life Sciences*, 3:100057, 2023.

[64] Pei Wang, Xiangzheng Deng, and Sijian Jiang. Global warming, grain production and its efficiency: Case study of major grain production region. *Ecological indicators*, 105:563–570, 2019.

[65] Jingfeng Yang, Hongye Jin, Ruixiang Tang, Xiaotian Han, Qizhang Feng, Haoming Jiang, Shaochen Zhong, Bing Yin, and Xia Hu. Harnessing the power of llms in practice: A survey on chatgpt and beyond. *ACM Transactions on Knowledge Discovery from Data*, 18(6):1–32, 2024.

[66] Shanglong Yang, Zhipeng Yuan, Shunbao Li, Ruoling Peng, Kang Liu, and Po Yang. Gpt-4 as evaluator: Evaluating large language models on pest management in agriculture. *arXiv preprint arXiv:2403.11858*, 2024.

[67] Di Zhang, Wei Liu, Qian Tan, Jingdan Chen, Hang Yan, Yuliang Yan, Jiatong Li, Weiran Huang, Xiangyu Yue, Dongzhan Zhou, et al. Chemllm: A chemical large language model. *arXiv preprint arXiv:2402.06852*, 2024.

[68] Shengyu Zhang, Linfeng Dong, Xiaoya Li, Sen Zhang, Xiaofei Sun, Shuhe Wang, Jiwei Li, Runyi Hu, Tianwei Zhang, Fei Wu, et al. Instruction tuning for large language models: A survey. *arXiv preprint arXiv:2308.10792*, 2023.

[69] Huaixiu Steven Zheng, Swaroop Mishra, Xinyun Chen, Heng-Tze Cheng, Ed H Chi, Quoc V Le, and Denny Zhou. Take a step back: Evoking reasoning via abstraction in large language models. *arXiv preprint arXiv:2310.06117*, 2023.

[70] Yaowei Zheng, Richong Zhang, Junhao Zhang, Yanhan Ye, Zheyan Luo, and Yongqiang Ma. Llamafactory: Unified efficient fine-tuning of 100+ language models. *arXiv preprint arXiv:2403.13372*, 2024.

# Appendix

# A    Data and Code Availability

## A.1    Code

The code for prompting and training is available at `https://github.com/open-sciencelab/Crop`.

## A.2    Data

The CROP dataset is available at `https://drive.google.com/drive/folders/11be8-Gd3h_VntRhmiIdhwopiAS8fh5Y7?usp=sharing`.

The CROP benchmark is available at `https://drive.google.com/drive/folders/11be8-Gd3h_VntRhmiIdhwopiAS8fh5Y7?usp=sharing`.

## A.3    Result

The test results of API models (GPT-3.5, GPT-4, Claude-3-Opus, Qwen-max) are available at `https://github.com/open-sciencelab/Crop`.

# B    Dataset Documentation

We adhere to the established dataset schemas as delineated in [28], with the objective of furnishing comprehensive information about our dataset to the prospective dataset consumers.

## B.1    Motivation

**For what purpose was the dataset created?**    The impetus for the development of the dataset is to augment the proficiency of Large Language Models in knowledge-intensive application scenarios, specifically within the realm of crop science. Existing datasets exhibit limitations in adequately encapsulating these knowledge-intensive areas or may not present the information in a manner that is readily interpretable by LLMs. By curating this novel dataset, we can facilitate the training of LLMs to comprehend and respond more effectively to queries in these areas. This could culminate in more precise and informative responses, thereby enhancing decision-making processes in crop science. Moreover, the dataset can serve as an invaluable resource for researchers and practitioners in the field, offering a rich repository of information and insights. It also holds the potential to propel the advancement of AI applications in agriculture, fostering more sustainable and efficient crop farming practices.

**Who created the dataset (for example, which team, research group) and on behalf of which entity (for example, company, institution, organization)?**    Researchers in the Shanghai AI Laboratory created the dataset.

**Who funded the creation of the dataset?**    Shanghai AI Laboratory, Yazhouwan National Laboratory, China Agricultural University.

## B.2    Composition

**What do the instances that comprise the dataset represent (for example, documents, photos, people, countries)?**    Each instance is the question-answer pair.

**How many instances are there in total (of each type, if appropriate)?**    See Table 4 and Table 5.

**Does the dataset contain all possible instances or is it a sample (not necessarily random) of instances from a larger set?**    Our dataset is composed of question-answer pairs about rice and corn. This contains a subset of crops. We have applied manual filtering to find related content.

**What data does each instance consist of?**    Each question-answer pair in our dataset consists of a question and an answer, where reference material is provided for open-book questions.

**Is there a label or target associated with each instance?**     We group each instance into a specific task.

**Is any information missing from individual instances?**     N/A.

**Are relationships between individual instances made explicit (for example, users' movie ratings, social network links)?**     The relationship between individual instances can be determined by the defined task.

**Are there recommended data splits (for example, training, development/validation, testing)?**     In this paper, we propose a dataset as well as a benchmark, thus there is no need to fix a recommended data split.

**Are there any errors, sources of noise, or redundancies in the dataset?**     Yes, there are. See Table 4.

**Is the dataset self-contained, or does it link to or otherwise rely on external resources (for example, websites, tweets, other datasets)?**     The dataset is self-contained for the tasks described in the paper.

**Does the dataset contain data that might be considered confidential (for example, data that is protected by legal privilege or by doctor–patient confidentiality, data that includes the content of individuals' non-public communications)?**     No.

**Does the dataset contain data that, if viewed directly, might be offensive, insulting, threatening, or might otherwise cause anxiety?**     No.

**Does the dataset identify any subpopulations (for example, by age, gender)?**     No.

**Is it possible to identify individuals (that is, one or more natural persons), either directly or indirectly (that is, in combination with other data) from the dataset?**     No.

**Does the dataset contain data that might be considered sensitive in any way (for example, data that reveals race or ethnic origins, sexual orientations, religious beliefs, political opinions or union memberships, or locations; financial or health data; biometric or genetic data; forms of government identification, such as social security numbers; criminal history)?**     See Appendix C.

### B.3   Collection Process

**How was the data associated with each instance acquired? Was the data directly observable (for example, raw text, movie ratings), reported by subjects (for example, survey responses), or indirectly inferred/ derived from other data (for example, part-of-speech tags, model-based guesses for age or language)?**     See Section 3.2.

**What mechanisms or procedures were used to collect the data (for example, hardware appara tuses or sensors, manual human curation, software programs, software APIs)?**     See Section 2 and 4.

**If the dataset is a sample from a larger set, what was the sampling strategy (for example, deterministic, probabilistic with specific sampling probabilities)?**     N/A.

**Who was involved in the data collection process (for example, students, crowdworkers, con tractors) and how were they compensated (for example, how much were crowdworkers paid)?**     The authors and their institutes were involved in the data collection process.

**Over what timeframe was the data collected?**     The data collection was conducted from January 2024 to May 2024.

**Were any ethical review processes conducted (for example, by an institutional review board)?** N/A.

**Did you collect the data from the individuals in question directly, or obtain it via third parties or other sources (for example, websites)?** See Section 3.2.

**Were the individuals in question notified about the data collection?** N/A.

**Did the individuals in question consent to the collection and use of their data?** N/A.

**If consent was obtained, were the consenting individuals provided with a mechanism to revoke their consent in the future or for certain uses?** N/A.

**Has an analysis of the potential impact of the dataset and its use on data subjects (for example, a data protection impact analysis) been conducted?** See Appendix C.

### B.4 Preprocessing/Cleaning/Labeling

**Was any preprocessing/cleaning/labeling of the data done (for example, discretization or bucketing, tokenization, part-of-speech tagging, SIFT feature extraction, removal of instances, processing of missing values)?** Yes. See Section 2.3 and Section 2.4.

**Was the "raw" data saved in addition to the preprocessed/cleaned/ labeled data (for example, to support unanticipated future uses)?** Yes.

**Is the software that was used to preprocess/clean/label the data available?** Not yet available.

### B.5 Uses

**Has the dataset been used for any tasks already?** No.

**Is there a repository that links to any or all papers or systems that use the dataset?** See Appendix A.

**What (other) tasks could the dataset be used for?** The dataset can also be used to pre-train an LLM for professional knowledge acquisition or fine-tune a multi-modal model for instruction following.

**Is there anything about the composition of the dataset or the way it was collected and preprocessed/cleaned/labeled that might impact future uses?** No.

**Are there tasks for which the dataset should not be used?** See Appendix C.

### B.6 Distribution

**Will the dataset be distributed to third parties outside of the entity (for example, company, institution, organization) on behalf of which the dataset was created?** Yes.

**How will the dataset be distributed (for example, tarball on website, API, GitHub)?** See Appendix A.

**When will the dataset be distributed?** The dataset is already available at `https://drive.google.com/drive/folders/11be8-Gd3h_VntRhmiIdhwopiAS8fh5Y7?usp=sharing`.

**Will the dataset be distributed under a copyright or other intellectual property (IP) license, and/or under applicable terms of use (ToU)?** See Appendix A. Except for third-party works, the extent to which they are attributed to the authors is distributed under a specific license that can be found under the dataset repository.

**Have any third parties imposed IP-based or other restrictions on the data associated with the instances?** The third-party data is not for commercial use and is otherwise subject to the terms and conditions of the organization.

**Do any export controls or other regulatory restrictions apply to the dataset or to individual instances?** No.

### B.7 Maintenance

**Who will be supporting/hosting/maintaining the dataset?** The authors will be.

**How can the owner/curator/manager of the dataset be contacted (for example, email address)?** By email address.

**Is there an erratum?** No.

**Will the dataset be updated (for example, to correct labeling errors, add new instances, delete instances)?** It is conceivable that modifications could be made to the dataset within the codebase, either for the rectification of inaccuracies or for the addition or removal of instances, contingent upon specific requests.

**If the dataset relates to people, are there applicable limits on the retention of the data associated with the instances (for example, were the individuals in question told that their data would be retained for a fixed period of time and then deleted)?** N/A.

**Will older versions of the dataset continue to be supported/hosted/maintained?** It depends on the nature of the dataset update. We may notify the dataset consumers via a website or codebase.

**If others want to extend/augment/build on/contribute to the dataset, is there a mechanism for them to do so?** Yes. One can use the proposed pipeline to generate new data. But any additional work should be considered as a distinct contribution, independent of our original work.

## C   Ethics Statement

In alignment with the NeurIPS Code of Ethics [5], we offer the following statement on the data-related concerns and potential societal impacts of our research.

### C.1   Data-related Concerns

**Privacy.** Our dataset comprises text from research articles, professional books, crop enterprise reports, and structural data. Since enterprise reports may contain sensitive information, we clean the original data before generating Q&A pairs to ensure privacy.

**Consent.** Our dataset includes information such as corporate disclosures, and we recognize that the information is intended to be publicly available.

**Deprecated Datasets.** Not applicable. We do not use any deprecated datasets.

**Copyright and Fair Use.** Because the dataset in this paper is original, the authors bear all responsibility in case of violation of rights and confirm that our dataset is open-sourced under the CC BY NC 4.0 license.

**Representative Evaluation Practice.**    Our dataset primarily considers two types of crops: rice and corn. In Appendix G, Table 4 and 5 list the main types and tasks included. Consequently, we have successfully constructed comprehensive domain-focused and general question-answer pairs for rice and corn. However, we must acknowledge that our dataset faces several limitations. Users should be aware of the content discussed in D.

**Labor Compensation.**    Regarding the ethical concern about fair compensation for domain experts and laborers, we would like to clarify that these individuals volunteered to contribute to the project. Their motivation stems from a desire to support the open-source community and raise awareness about the crop science domain. As such, there is no specific price for single-turn or multi-turn dialogues. We created the first version of the CROP suite, and view this as a starting point. We aim to engage the academic community in future iterations of CROP and are grateful for the efforts of involved domain experts and laborers who have contributed thus far.

## C.2   Potential Social Impacts

**Safety.**    Our technology cannot be used to harm individuals.

**Security.**    The starting point of our dataset is to provide guidance for current crop science techniques. However, some of the knowledge is derived from historical experience. To avoid security problems such as blindly referencing this information and potentially impacting agricultural production, users of our dataset should consider it comprehensively.

**Discrimination.**    Not applicable. Our dataset is targeted at crop science.

**Surveillance.**    Not applicable. Our dataset is targeted at crops rather than individuals.

**Deception & Harassment.**    We believe there is a low likelihood that this dataset could be used to facilitate deceptive interactions.

**Environment.**    As a dataset intended for application in agricultural production, our starting point is to utilize limited land resources to increase food yield and address hunger issues. Despite our careful efforts in creating the dataset, such as extensive manual screening of the data, there may still be misleading information present. This misleading information could impact crop science, such as inappropriate levels of pesticide use, thereby causing environmental burdens. We sincerely hope that our dataset will have a predominantly positive impact, which requires researchers to consider it thoroughly when using this dataset.

**Human Rights.**    Not applicable. Our dataset is targeted at crops.

**Bias and fairness.**    We performed extensive manual screening during the creation of our dataset, therefore the likelihood of biases related to specific races, genders, *etc*., is very low.

# D   Limitations

## D.1   Raw Data

The comprehensiveness of the raw data is somewhat limited, given that the discussion is confined to a select number of crop species. Corn and rice are considered representative examples, notwithstanding the existence of other prevalent crop families such as wheat, tuber crops, and beans, which were rarely incorporated in the raw data collection. Even within the context of corn and rice, the raw data may exhibit a bias towards specific varieties or geographical regions. For instance, research papers in Chinese predominantly focus on varieties of corn or rice cultivated in mainland China.

In terms of the format of the source data, text was the sole modality considered during the collection of raw data, with no inclusion of images, audio, or videos. One of the primary motivations for this approach was to ensure compatibility with the GPT-4 API in the dataset generation pipeline. However,

images or charts omitted from research papers could potentially provide additional context for the formulation of Question-Answer pairs if included.

Crop science is a dynamically evolving field, and as such, we do not intend for this initial collection of raw data to remain aligned with the most current information. While the most recent research study within the raw data was published in 2022, new findings can be integrated to augment the dataset, with the entire data generation pipeline readily available.

## D.2 Dataset Construction Pipeline

We acknowledge that the origin of errors can be traced back to the selected extraction tools, prompting substantial efforts towards enhancing the quality of the dataset. As one of the meticulously chosen information extraction tools, despite being effective in recognizing texts within documents, [46] demonstrated sub-optimal performance when dealing with all forms of charts. Other presented extraction tools also exhibited varying degrees of information loss.

With a reference context given, an LLM could manifest certain biases when generating Q&A pairs. The model might interpret the context in a biased manner based on its training data, namely the generation bias. For instance, if the model has been trained on data that frequently associates certain practices with specific crop varieties, it might overgeneralize these associations in the generated Q&A pairs. In some cases, the LLM may not fully consider the provided context when generating Q&A pairs, leading to answers that might be inconsistent with the context.

Tasks are defined with guidance from researchers and domain experts. This can lead to a bias towards certain types of tasks that involve individuals deemed important. Due to our task design settings, we also do not expect an LLM fine-tuned using the proposed dataset to perform significantly better in any task regarding scopes other than Information Access.

## D.3 Benchmark and Evaluation

Recent advancements in the field of LLMs predominantly feature models with around 70 billion or 110 billion parameters. The selection of these specific parameter sizes represents a strategic equilibrium between enhancing model performance and efficiently allocating computational resources. Further empirical investigations are necessitated to ascertain the efficacy of the suggested dataset in the fine-tuning of LLMs with these parameter dimensions.

# E Related Work

## E.1 AI for Agriculture

The integration of Artificial Intelligence (AI) into various sectors has led to transformative advancements in recent years, with agriculture being no exception [9]. As a labor-intensive field, agriculture must adapt to the growing population and the escalating demands for crop production, making automation an essential advancement [33]. AI stands to contribute by alleviating the burden of labor-intensive tasks and enhancing both the yield and quality of agricultural products, thereby ensuring a stable global food supply [54, 63].

On the one hand, the advanced robotic technologies incorporated with AI, are playing a vital role in tackling the labor-intensive issue while ensuring the crop-producing process [44, 22]. Robotics, such as wheeled vehicles [41] equipped with RGB cameras and laser sensors, by utilizing AI-Vision algorithms, are adept at land preparation before planting [4, 35], sowing [31, 56], plant treatment [52, 27], and crop harvesting [13]. In addition, the implementation of Unmanned Aerial Vehicles (UAVs) [19, 61] and Internet of Things (IoTs) [51] can build a three-dimensional monitoring system for crop cultivation, thus promising significant reduction in cost and increase in the yield [60].

On the other hand, advanced AI can enhance crop cultivation in a data-driven approach, achieving the goals of precision agriculture [53, 39]. The combination of sensor data from equipment, along with remote sensing technologies like satellites and drones, soil and weather data, and other available data sources, can generate a dataset abundant in contextual information [47]. Consequently, processes such as irrigation, fertilization, and pest control in crop cultivation can be refined through the timely predictions provided by sophisticated AI methods [14, 62].

The introduction of LLMs has provided crop science researchers and agricultural practitioners with new opportunities to enhance production [50, 37]. The robust ability of LLMs to generate and acquire domain-specific knowledge has been a significant factor in this potential [17]. While researchers have explored the use of LLMs in answering agriculture-related exams [55], their performance in certain crop cultivation scenarios, such as pest management, has been less than satisfactory [66]. Moreover, there remains a considerable gap between the ability to answer exam questions and the application of this knowledge in real-world situations. To bridge the gap and thoroughly assess LLMs in supporting the crop science field, we introduce CROP. CROP comprises an instruction tuning dataset that equips LLMs with the necessary skills to aid tasks in crop production, along with a carefully designed benchmark to evaluate the extent to which LLMs fulfill the demands of real-world agricultural applications. We anticipate that CROP will serve the research community and also provide practical benefits to industry practitioners.

## E.2 LLM-based Multi-turn Dialogue Generation

In recent research, several LLM-based approaches have emerged for constructing multi-turn dialogues. [21] tried to prompt LLMs to generate full multi-turn dialogues based on the 'ChatSEED', which were sampled from the very first utterances of real-world human dialogues. [57] first trained a question-asking model and then collected multi-turn conversations by utilizing it to iteratively chat with ChatGPT. [45] also used a two-stage approach, where instructional strategies were first extracted from real human-machine dialogues and then applied to produce new instructions. While these approaches collectively contribute to advancing LLM-based multi-turn dialogue systems, they are either restricted in terms of context dependency or less likely to be transformed into an end-to-end system.

## F Dataset Collection Detail

We provide more details of the experts' evaluation. The evaluation work of human experts on single-turn and multi-turn dialogues is based on the same metrics designed for the LLM grader. A total of 115 participants were involved, including 14 domestic rice breeding experts, 49 graduate students in breeding-related fields, and 52 undergraduate students (junior year and above). For single-turn dialogues, experts focus on correctness, professionalism, and richness, while for multi-turn dialogues, experts focus on coherence, consistency, and specificity. After the first stage of filtering work, the generated contents include around 220k single-turn dialogues and 2k multi-turn dialogues. Our human experts clean the generated contents based on the mentioned rules, deleting around 5% low-quality samples of single-turn dialogues and 7% low-quality samples of multi-turn dialogues. After the filtering by human experts, we have 210,038 samples of single-turn dialogues and 1871 samples of multi-turn dialogues.

## G Dataset Analysis Detail

### G.1 Basic Statistics

Table 4 and 5 show the composition of single-turn dialogues and multi-turn dialogues, respectively.

### G.2 Diversity

**Data source.** Figure 6 and 7 show the data source distribution of rice and corn, respectively.

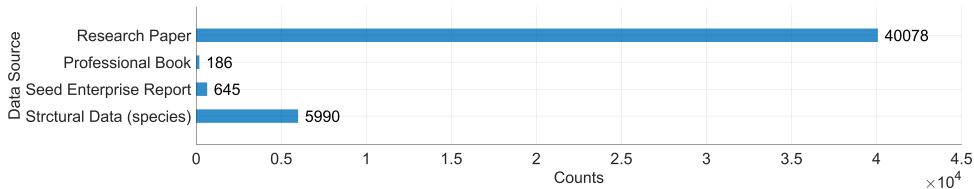

Figure 6: Distribution of rice-related data source. Counts include both Chinese and English.

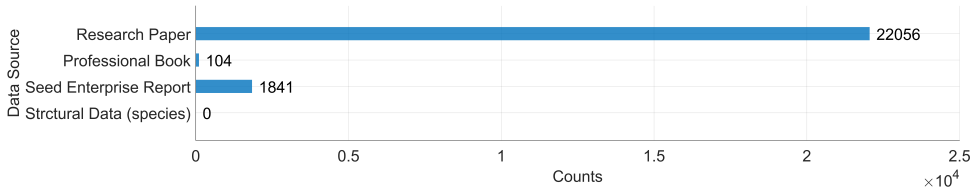

Figure 7: Distribution of corn-related data source. Counts include both Chinese and English.

Table 4: Composition of single-turn dialogues. Please note that despite our data-cleaning efforts, the final CROP dataset inevitably contains a small amount of data (<0.5%) from other grains like wheat. As this portion does not dominantly influence the fine-tuning results, it is included in the final CROP dataset. We have listed it explicitly in the table to avoid any misleading counts.

| Cereal | Type | Task | Abbr. | English Q&A | Chinese Q&A | Total |
|---|---|---|---|---|---|---|
| Rice | Domain | Closed-book Q&A | CQA | 42,951 | 83,396 | 126,347 |
| | | Open-book Q&A | OQA | 2,430 | 2,037 | 4,467 |
| | General | Event Extraction | EE | 1,891 | 1,030 | |
| | | Named Entity Recognition | NER | 2,003 | 1,604 | 9,742 |
| | | Summary | Summary | 1,586 | 1,628 | |
| Corn | Domain | Closed-book Q&A | CQA | 25,259 | 27,667 | 52,926 |
| | | Open-book Q&A | OQA | 3,202 | 3,047 | 6,249 |
| | General | Event Extraction | EE | 2,245 | 1,322 | |
| | | Named Entity Recognition | NER | 2,008 | 1,316 | 10,307 |
| | | Summary | Summary | 1,559 | 1,857 | |
| Others* | | — | | | | <1000 |
| Overall | | — | | 85,134 | 124,904 | **210,038** |

Table 5: Composition of multi-turn dialogues. Blue denotes counts of 3-turn dialogues, green denotes counts of 4-turn dialogues, and yellow denotes counts of 5-turn dialogues.

| Cereal | Scenario | Task | English Q&A | Chinese Q&A | Total |
|---|---|---|---|---|---|
| Rice | Problem Solving | Pest Control | 14+71 | 8+37 | 130 |
| | | Nutrient Supplementation | 19+93 | 2+90+1 | 205 |
| | | Disease Containment | 19+60 | 4+39 | 122 |
| | Personalized Recommendation | Crop Variety Selection | 12+53 | 9+9 | 83 |
| | | Resource Management | 4+110+1 | 5+50 | 170 |
| | Knowledge Interpretation | Research Interpretation | 3+125+1 | 8+85 | 222 |
| Corn | Problem Solving | Pest Control | 20+84 | 7+77 | 188 |
| | | Nutrient Supplementation | 24+56 | 8+30 | 118 |
| | | Disease Containment | 21+64 | 2+19+1 | 107 |
| | Personalized Recommendation | Crop Variety Selection | 19+75 | 46+47 | 187 |
| | | Resource Management | 8+94 | 1+69 | 172 |
| | Knowledge Interpretation | Research Interpretation | 5+94+1 | 6+61 | 167 |
| Overall | | — | 1,150 | 721 | **1,871** |

**Geographic Diversity.** Table 6 and 7 show the geographic diversity of rice and corn data, respectively.

Table 6: Geographic diversity of rice dataset. Note that we only include countries with more than 5 occurrences in all samples.

| Continent | Country |
|---|---|
| Asia | Bangladesh, China, India, Indonesia, Japan, Laos, Nepal, North Korea, Philippines, South Korea, Thailand, Vietnam |
| Europe | France, Germany, Hungary, Italy, Russia, Spain |
| North America | Cuba, Guatemala, Mexico, United States |
| South America | Argentina, Brazil, Colombia, Guyana |
| Africa | Egypt, Guinea, Kenya, Madagascar, Mali, Tanzania |
| Oceania | Australia |

Table 7: Geographic diversity of corn dataset. Note that we only include countries with more than 5 occurrences in all samples.

| Continent | Country |
|---|---|
| Asia | Bangladesh, China, India, Indonesia, Japan, Laos, Nepal, Philippines, South Korea, Thailand |
| Europe | France, Germany, Greece, Hungary, Italy, Malta, Poland, Romania, Russia, Serbia, Spain, United Kingdom |
| North America | Canada, Cuba, Guatemala, Mexico, United States |
| South America | Argentina, Brazil, Colombia, Guyana, Peru |
| Africa | Congo, Egypt, Guinea, Kenya, Madagascar, South Africa, Tanzania, Zambia, Zimbabwe |
| Oceania | Australia, New Zealand |

**Content Length.** Table 8 shows the statistics of content length measured by token counts of single-turn dialogue datasets. We use the official GPT-4 tokenizer provided by OpenAI. Figure 8 shows the statistics of the number of rounds in multi-turn dialogues.

**Content Distribution.** Figure 9 and 10 show the information cartography of the Chinese and English Q&A instructions, respectively.

## H  Template for Structural Data

Inspired by [67], we use the template technique to transform structured data into a dialogue format suitable for fine-tuning LLMs. There are 5 templates in different styles provided for two properties, presented in Table 9 and 10. We translate the original Chinese template into English as we are provided with the structured data in Chinese for a better illustration. By inserting the name of a rice variety into "[Rice Species]" and providing answers in simple declarative sentences, complete Q&A pairs can be obtained.

## I  Benchmark Analysis Detail

### I.1  Topic Distribution

Figure 11 shows the topic distribution of the CROP benchmark.

Table 8: Statistics of token counts of single-turn dialogues.

| Cereal | Language | Task | Max Q token | Max A token | Avg Q token | Avg A token |
|--------|----------|---------|-------------|-------------|-------------|-------------|
| Rice | EN | CQA | 45 | 179 | 18.79 | 62.11 |
| | | OQA | 262 | 114 | 89.54 | 44.47 |
| | | EE | 586 | 148 | 98.57 | 46.22 |
| | | NER | 386 | 106 | 108.97 | 21.99 |
| | | Summary | 1480 | 194 | 203.06 | 99.20 |
| | ZH | CQA | 191 | 1779 | 27.38 | 123.55 |
| | | OQA | 1100 | 425 | 235.89 | 105.36 |
| | | EE | 770 | 192 | 209.33 | 80.08 |
| | | NER | 1135 | 215 | 277.74 | 57.00 |
| | | Summary | 2059 | 713 | 384.01 | 175.40 |
| Corn | EN | CQA | 56 | 203 | 22.36 | 86.68 |
| | | OQA | 377 | 203 | 91.76 | 43.94 |
| | | EE | 785 | 123 | 105.42 | 48.88 |
| | | NER | 329 | 184 | 108.26 | 22.01 |
| | | Summary | 2183 | 189 | 213.53 | 96.62 |
| | ZH | CQA | 138 | 500 | 31.47 | 139.91 |
| | | OQA | 1213 | 443 | 255.94 | 103.23 |
| | | EE | 921 | 170 | 206.33 | 74.94 |
| | | NER | 3194 | 570 | 362.09 | 53.94 |
| | | Summary | 3188 | 314 | 440.65 | 172.26 |

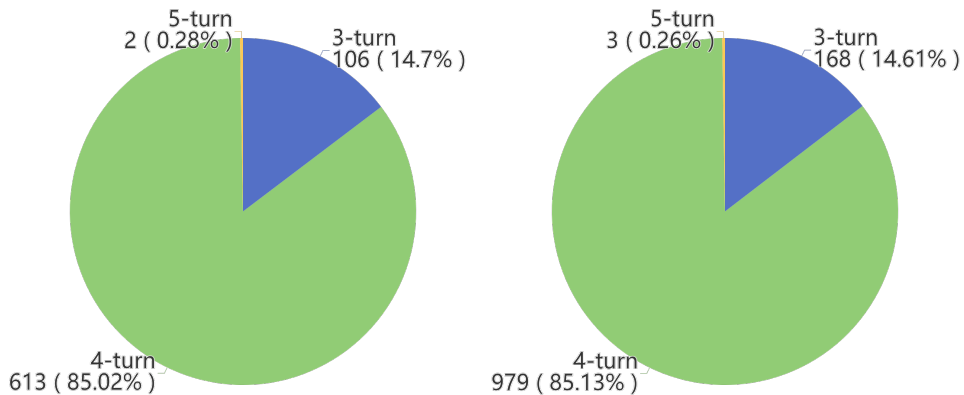

(a) Chinese Q&A.  (b) English Q&A.

Figure 8: Statistics on the number of rounds in multi-turn dialogues. Both for Chinese Q&A and English Q&A, the majority is 4-turn dialogues.

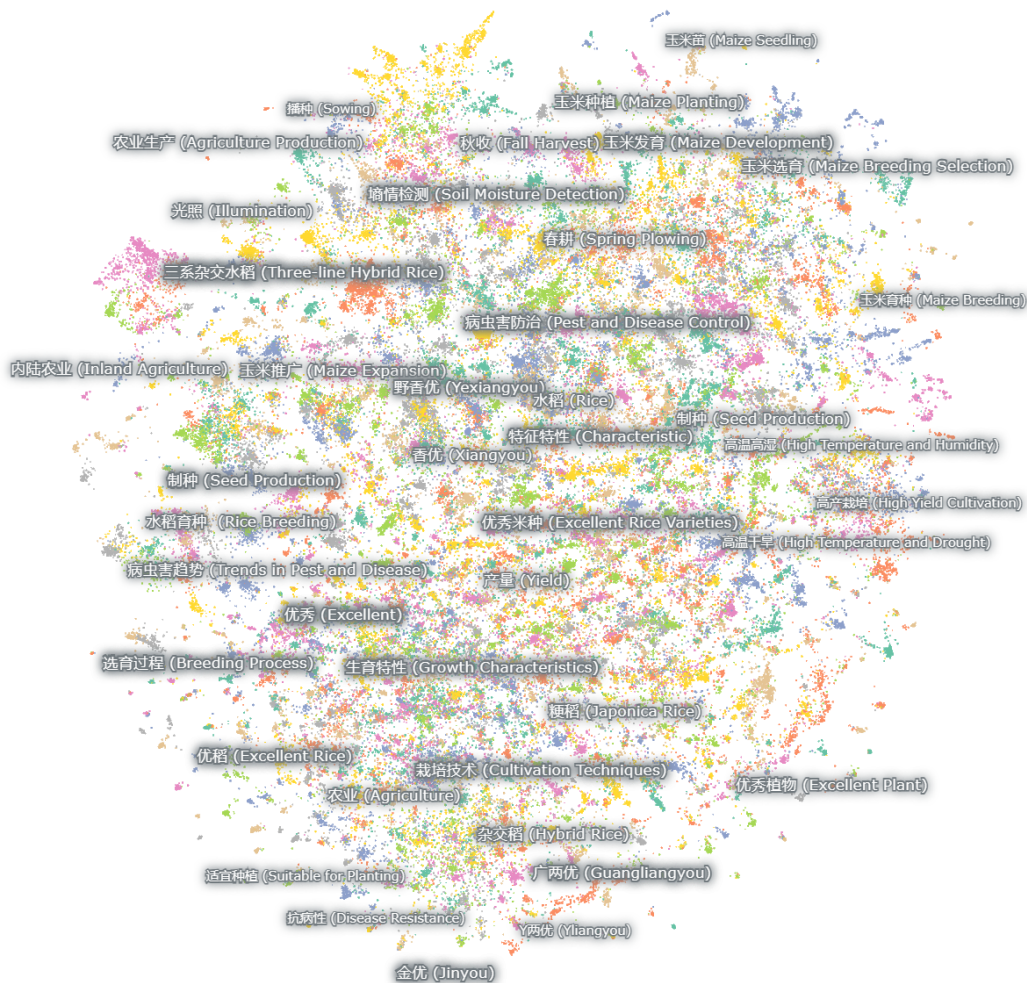

Figure 9: Chinese Q&A Instructions Cartography. We use the Chinese Q&A including the closed-book domain Q&A and open-book domain Q&A for generating the information cartography, where multiple diverse topics are revealed. Chinese topics and corresponding translations are presented and the map was generated using Nomic Atlas.

Table 9: 5 Templates in different styles for the rice property "Rice Blast Grade". We translate the original Chinese template into English.

| Style | Template |
|---|---|
| Simple Inquiry | "What is the Rice Blast Grade for [Rice Species]?" |
| Detailed Request | "Could you please provide the Rice Blast Grade for [Rice Species], including any relevant details on lesion coverage and severity?" |
| Expert Opinion | "As an expert, what Rice Blast Grade would you give to [Rice Species] based on the severity and spread of the lesions?" |
| Historical Data | "Based on historical agricultural data, which Rice Blast Grade was assessed on [Rice Species]?" |
| Comprehensive Evaluation | "Based on the current conditions and symptoms observed, what would be the appropriate Rice Blast Grade for [Rice Species]?" |

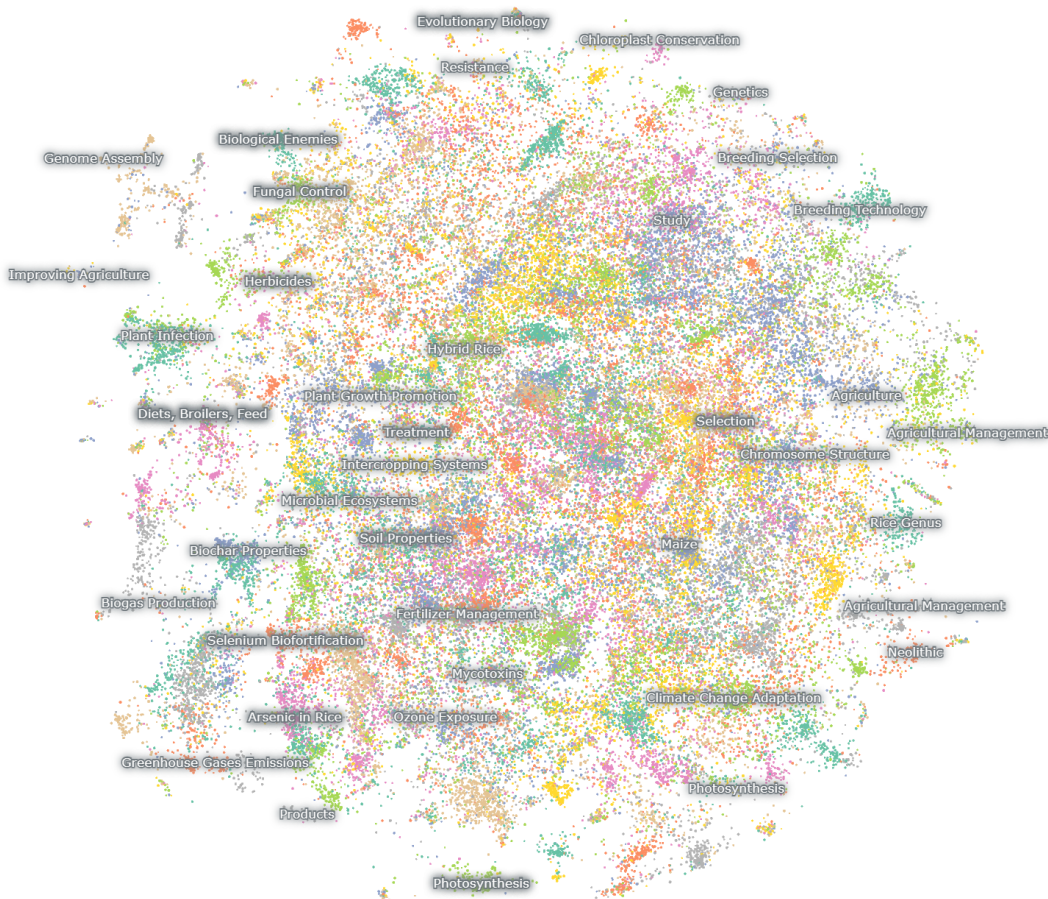

Figure 10: English Q&A Instructions Cartography. We use the English Q&A including the closed-book domain Q&A and open-book domain Q&A for generating the information cartography, where multiple diverse topics are revealed. The map was generated using Nomic Atlas.

Table 10: 5 Templates in different styles for the rice property "Suitable Cultivation Area". We translate the original Chinese template into English.

| Style | Template |
|---|---|
| Simple Inquiry | "Which area is suitable for [Rice Species]'s cultivation?" |
| Detailed Request | "Could you provide detailed information on the regions that are most suitable for planting [Rice Species], considering soil quality and climate conditions?" |
| Expert Opinion | "As an agricultural expert, which areas would you recommend for planting [Rice Species]?" |
| Historical Data | "Based on historical agricultural data, which areas have consistently proven to be suitable for [Rice Species]'s cultivation?" |
| Comprehensive Evaluation | "Please provide a comprehensive evaluation of the best areas for planting [Rice Species], considering factors such as soil, climate, and water resources." |

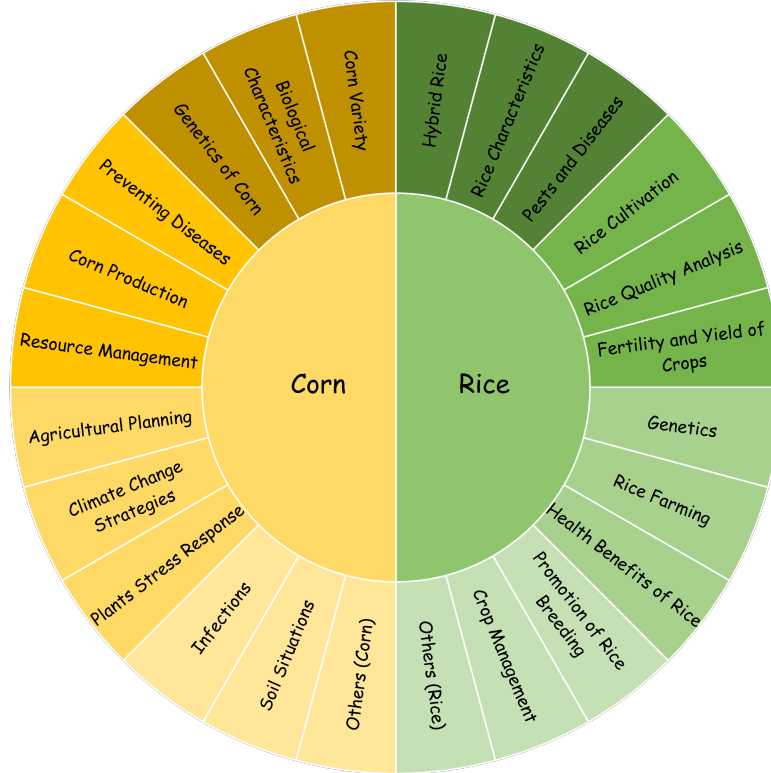

Figure 11: Topic distribution of benchmark. We list the keywords in the CROP benchmark for a deeper insight. Darker colors indicate a higher frequency of occurrence, while lighter colors indicate a lower frequency of occurrence.

## I.2 Benchmark Comparison

Table 11: Geographic diversity of benchmark. Note that we only record countries with more than 3 occurrences in benchmark samples.

| Continent | Country |
|---|---|
| Asia | Bangladesh, China, India, Indonesia, Japan, Nepal, North Korea, South Korea, Thailand, Vietnam |
| Europe | Germany, Italy, Russia, Spain |
| North America | Cuba, Mexico, United States |
| South America | Brazil |
| Africa | Madagascar, Mali, South Africa |
| Oceania | Australia |

Table 11 shows the geographic diversity of the CROP benchmark. The statistical rules we adopted are the same as those applied to the CROP dataset (See 3.2), except the occurrence threshold was adjusted more than 3 times. There are 22 countries across six continents included in the CROP benchmark.

Table 12 compares the CROP benchmark against other agriculture-related question databases across four dimensions: language, format, dataset size, and region. CROP benchmark includes questions in two major world languages, English and Chinese, unlike other benchmarks limited to a single language. In terms of volume and geographic coverage, CROP benchmark significantly outperforms the other three benchmarks, offering roughly triple the number of questions found in AgriExams and

Table 12: Benchmark Comparison. CROP benchmark surpasses existing datasets in terms of quantity and locality.

| Dataset | Language | Format | Size | Region |
|---|---|---|---|---|
| Certified Crop Advisor (CCA) Exam[1] | English | MCQs | 312 | United States |
| EMBRAPA[2] | Portuguese | Test-based Inquires | 1,000 | Brazil |
| AgriExams[3] | English | MCQs | 1,723 | India |
| CROP (Ours) | English & Chinese | MCQs | 5,045 | 22 Countries |

[1] https://www.certifiedcropadviser.org/become-certified/certifications/.
[2] https://mais500p500r.sct.embrapa.br/view/index.php. For EMBRAPA, we count the number of test-based inquiries related to rice and corn.
[3] https://www.agriexam.com/agriculture-previous-year-question-paper.

spanning 22 countries. Clearly, CROP benchmark is equipped to provide a more extensive evaluation of LLMs.

## J Tasks and Prompts

### J.1 Dual-guided Task Design

We propose a prior-based, dual guidance enhanced framework that integrates insights from both researchers and domain experts into the task design workflow. Ultrachat [20] proposed three scopes of dialogue data: Information Access, Information Creation, and Information Transformation, to model human-AI interactions. While the scope dimension poses a certain level of restriction on the task design, additional variables are required to define a complete task partitioning criterion.

We consider domain $D$ and raw data $R$ as two dimensions which, together with scope $S$, help determine a task sample space. The prior is thus defined as follows:

$$K_{prior} = (d, s, r), \quad d \in D, s \in S, r \in R, \tag{7}$$

where we consider $d$ being the crop science domain, $s$ being either the Information Access scope or the Information Transformation scope, and $r$ being the data source.

For single-turn dialogues, we delineated two categories of tasks under the guidance of researchers and domain experts. Question-answer pairs in the domain-focused category belong to what is generally called "knowledge Q&A". An LLM fine-tuned using this portion of the single-turn dialogues is expected to provide highly specialized and well-formatted responses to domain knowledge queries. The general category aims to adapt general language modeling tasks to the crop science domain. Each task within this category is domain-adapted, thereby further enhancing the capabilities of an equipped LLM.

A task and its setting are determined separately. Within the task sample space defined by a prior triplet $(d, s, r)$, subdivided tasks $T$ of a category are determined based on researcher guidance $G_R$ and expert guidance $G_E$:

$$T \sim P^{ST}(t|G_R, G_E, K_{prior}) \tag{8}$$

A specific setting $\tau$, is chosen from a set of possible settings $\tau_t$ for a given task $t$, which is influenced by additional expert guidance $G'_E$:

$$\tau_t \sim P^{ST}(\tau|t, G'_E) \tag{9}$$

Consider NER as a task $t$ within the open-book category, additional expert guidance $G'_E$ specifies entity categories to be geographic location, disease, pest, and crop variety as a task setting $\tau$. We intend to enhance the capability of an LLM in recognizing selected entity types in crop science.

As discussed, tasks for multi-turn dialogues are also defined in a two-step manner, with recommended scenarios first and tasks afterward. The work of determining concrete tasks is assigned to an LLM with contextual prompts from raw data.

Within the task sample space defined by a prior triplet $(d, s, r)$, the set of scenarios $S$ is determined based on only expert guidance $G_R$:

$$S \sim P^{MT}(s|G_E, K_{prior}) \tag{10}$$

Tasks $T_s$ of a chosen scenario $s$ is generated using an LLM $L$:

$$T_s \sim P_L^{MT}(t|s) \tag{11}$$

Consider pest control as a task $t$ within the problem-solving scenario $s$. The LLM recognizes it as one of what should be included in a multi-turn setting as it is frequently discussed in the raw data.

The proposed framework is designed with high adaptability. The triplet of the three prior variables can be adjusted to cater to specific use cases, providing a flexible and customizable approach to task design.

## J.2 Single-turn Q&A

The single-turn Q&A includes five tasks:

**Closed-book Q&A.** The content of closed-book Q&A is professional knowledge, where there are no external reference materials provided in the question. Our Q&A pairs cover the following aspects: stress resistance, pests and diseases, cultivation techniques, breeding, characteristics, suitable planting areas, *etc*.

**Open-book Q&A.** The content of open-book Q&A is also professional knowledge, where there are external reference materials provided in the question. The considering aspects of the open-book Q&A are the same as closed-book Q&A.

**Summary Q&A.** Give a professional summary for a portion of the provided article.

**Named Entity Recognition.** Understand and extract specific entity information from the provided article. The entity information is emphasized on the geographic location, diseases and pests, and crop variety.

**Event Extraction.** Identify the event triggers and extract relevant arguments associated with these events from the provided article. The event categories restricted by the task are (1) no event, (2) pest impact, (3) new variety release or breeding, (4) climate or weather event, (5) planting or production trial, (6) policy release, (7) yield record, and (8) variety mutation.

We formulate the task-oriented prompts to query the LLM for the Q&A pairs and present several examples of produced Q&A pairs in this section, where the corresponding arrangements are shown in Table 13. We divided our prompt into four parts: role description, reference text, guidelines, and example dialogue.

Table 13: The arrangement of templates and examples for single-turn Q&A.

| EN | | ZH | |
|---|---|---|---|
| Task | Location | Task | Location |
| Closed-book Q&A | Figure 12 | Closed-book Q&A | Figure 13 |
| Summary Q&A | Figure 14 | Summary Q&A | Figure 15 |
| Named Entity Recognition | Figure 16 | Named Entity Recognition | Figure 17 |
| Event Extraction | Figure 18 | Event Extraction | Figure 19 |
| Open-book Q&A | Figure 20 | Open-book Q&A | Figure 21 |

**Closed-book Q&A (EN)**

**Topic Description:**
Instruct the model to return professional knowledge Q&A. There are no external reference materials provided in the produced question.

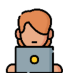

**System:**
You are an expert in the field of study relevant to the academic paper enclosed within the <doc></doc> tags. Please generate a set of question-answer pairs that reflect the key findings and discussions in the paper.

*Role Description*

<doc>
In the recent past, population explosion has augmented urbanization......
</doc>

*Reference Text*

For each question-answer pair, please adhere to the following guidelines:
1. The number of question-answer pairs should be determined by the depth and breadth of the paper, with a minimum of *N_min* and a maximum of *N_max*.
2. The questions should cover various aspects of the paper, avoid focusing solely on numerical data, and refrain from asking yes/no questions or questions with obvious answers.
3. If the paper provides relevant information, ensure that the question-answer pairs cover the following aspects: Stress Resistance, Pests and Diseases, Cultivation Techniques, Breeding, Characteristics, Suitable Planting Area, etc.
4. Ensure that the answers follow a logical sequence and are supported by the information in the paper.
5. You may refer to the example provided within the <example></example> tags for designing the question-answer pairs, but ensure that the output strictly follows the format of this example.

*Guidelines*

<example>
Question: How to achieve a high yield in the cultivation of Y302 through fertilizer management?
Answer: In the cultivation of Y302, high yield is achieved through the scientific ratio and timely application of base fertilizer, topdressing, ear fertilizer, and grain fertilizer. Considering the longer growth time and high yield potential of Y302, organic fertilizer needs to be added.
</example>

*Example Dialogue*

**User:**
Please generate a set of question-answer pairs based on the given academic paper.

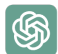

**Example:**
Question 1: What was the primary objective of the field research conducted on fourteen early genotypes of maize?
Answer 1: The primary objective of the field research was to explore the roles of chlorophyll content and its relation with other yield attributing traits in maize.
Question 2: What were the specific findings regarding aflatoxin levels in maize using different drying methods?
Answer 2: The research found significant differences in aflatoxin levels among the three drying methods tested. Kitchen drying resulted in the lowest aflatoxin contamination (1.4 ppb), followed by sun drying (1.6 ppb) and dry shelter drying (2.2 ppb). These findings indicate that kitchen drying is more effective at reducing aflatoxin levels in maize compared to the other methods.
Question 3: ……
Answer 3: ……

Figure 12: Prompt and few examples for English Closed-book Q&A.

**Closed-book Q&A (ZH)**

**Topic Description:**
Instruct the model to return professional knowledge Q&A. There are no external reference materials provided in the produced question.

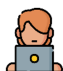

**System:**
你是一名育种专家，同时是一位农业科学和生物科学的教授，请基于在
<doc></doc>标记之间的这篇学术论文，生成若干问答对。

Role Description

<doc>
玉米营养诊断是在玉米生育进程中的某些时期检测植株的营养状况是否符合……
</doc>

Reference Text

对于每一个问答对，你需要注意以下几点：
1. 请根据论文长度决定生成问答对的具体数量，问题数量最小为$N\_min$，最大为$N\_max$。
2. 这些问题应当从不同的角度出发，不应当着重关注于数字部分，避免提出是/否的问题或有明显答案的问题。
3. 如果论文中有相关信息，请确保问答中包含涉及选育过程；特征特性；栽培技术要点；适宜种植区域；抗逆性；病虫害防治等方面的内容。
4. 请确保回答遵循合适的逻辑关系。
5. 你可以参照<example></example>标记之间的示例设计问答，但请确保完全遵循该示例的格式输出回复。

Guidelines

<example>
问题：需要AI Agent推荐一个水稻品种，适合在西藏等高海拔地区种植。
回答：中国高海拔地区通常是指海拔在500-4000米之间的地区，主要包括云南、贵州、四川、西藏等地。按照全国稻区划分，这些地区大多属于西南高原单双季稻稻作区，水稻面积占全国的8%左右。高海拔造成这些区域年积温低和极端低温，光照和紫外线强烈。同时，海拔决定水稻的垂直分布。云南省海拔1750 m以下的为籼稻，海拔1750～2000 m的为籼粳混交带，海拔2000 m以上的为粳稻。2023年西藏自治区农科院农研所选育的水稻新品系"2021LS－68"在海拔3177米实现亩产1112.32斤。此外，其他品种赣73优8601、赣宁粳1号等经过多年测试，能够适应高海拔种植。高海拔地区的气候条件差异可能会很大。因此，选择适宜种植品种，需要根据具体地区的平均光照温度报告或数据进行决策。
</example>

Example Dialogue

**User:**
请基于给定学术论文，生成若干问答对。

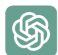

**Example:**
问题1：莱农糯10号玉米品种的主要特点是什么？
回答1：莱农糯10号玉米品种具有较短的生育期（75天），较高的抗病性，以及较高的支链淀粉含量（99.27%），适合在黄淮海夏玉米区种植。
问题2：桂糯518玉米品种的主要优势是什么？
回答2：桂糯518玉米品种具有较高的保绿度（88.9%），较低的倒伏率（2.5%），以及较高的产量表现，特别适合在广西种植。
问题3: ……
回答3: ……

Figure 13: Prompt and few examples for Chinese Closed-book Q&A.

**Summary (EN)**

**Topic Description:**
Instruct the model to return a professional summary for a portion of the provided article.

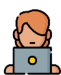

**System:**
You are an expert in the field of study relevant to the academic paper enclosed within the <doc></doc> tags. Please generate a set of question-answer pairs that summarize the key findings and discussions in the paper.

Role Description

<doc>
In the recent past, population explosion has augmented urbanization......
</doc>

Reference Text

For each question-answer pair, please adhere to the following guidelines:
1. The task specified for the question-answer pair is Summary.
2. Please extract a portion of the paper (at least 200 words) as the reference text for the task.
3. Please ensure that the answer follows the appropriate logical relationship.
4. You may refer to the example provided within the <example></example> tags for designing the question-answer pairs, but ensure that the output strictly follows the format of this example.
5. Please replace the part within '()' in the example question with the extracted reference text, and ensure not to change other parts of the question.

Guidelines

<example>
Question: (Folates, also known as vitamin B9, are one of the essential micronutrients for the human body. Due to the inability of the human body to synthesize on its own, it can only rely on food for intake. Adults need to consume at least 400 per day μ Folic acid is used to meet the needs of life activities, increasing to 600 during pregnancy μ G. But in many parts of the world, the daily intake of folic acid is around 400 μ As a result, folate deficiency has become a global health problem. Folate deficiency is more likely to worsen in pregnant and breastfeeding women. As the world's main cultivated crop and the largest grain crop in China, corn has an average folate content of 40-60% in dry seeds μ (g/100g) Only one tenth of the daily dietary requirement for folate in humans.) Please read and understand the given text, and summarize it.
Answer: This paragraph points out the importance of folic acid in human health and the problem of insufficient folic acid intake worldwide. Special emphasis was placed on the health issues that may arise from folic acid deficiency during pregnancy and lactation. At the same time, it was mentioned that the folate content of corn, the main crop, is generally low, which is different from the daily needs of the human body.
</example>

Example Dialogue

**User:**
Please generate *N* question-answer pairs based on the given academic paper.

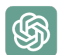

**Example:**
Question 1: (The water availability is calculated based on the amount of rainfall that occurs during the year. The rainfall per year suitable for corn cultivation is between 300 mm to 400 mm and 900 mm to 1200 mm. Based on the analysis of rainfall data from 2018 to 2022, the average This happens because of a monsoonal rainfall pattern in this study area. Hence, it is classified as not suitable (N) for the corn, as shown in Figure 4. The Dadahup irrigation area has low land elevation, so high rainfall may make the inundated land. Inundated land can inhibit the growth of corn. Based on the previous study in Kaliwungu Village, corn productivity with inundated land was 10.17 tons/ha. In comparison, corn production with regularly irrigated land was 12.10 tons/ha.) Read and understand the given text, and summarize it.
Answer 1: The passage discusses water availability for corn cultivation in the Dadahup irrigation area, emphasizing that the ideal annual rainfall for corn ranges from 300 mm to 1200 mm. However, due to the monsoonal rainfall pattern, the area experiences higher than ideal rainfall, leading to classification as unsuitable for corn cultivation. The low elevation of the land contributes to inundation, which further inhibits corn growth, as evidenced by lower productivity in inundated conditions compared to well-irrigated land.
Question 2: ……
Answer 2: ……

Figure 14: Prompt and few examples for English Summary Q&A.

**Summary (ZH)**

**Topic Description:**
Instruct the model to return a professional summary for a portion of the provided article.

**System:**
你是一名育种专家，同时是一位农业科学和生物科学的教授，请基于在
<doc></doc>标记之间的这篇学术论文，生成概括总结。

Role Description

<doc>
玉米营养诊断是在玉米生育进程中的某些时期检测植株的营养状况是否符合……
</doc>

Reference Text

对于每一个问答对，你需要注意以下几点：
1. 问答对的任务类型是概括总结。
2. 请提取文章的一部分（至少200字）作为问答对任务中的参考文本。
3. 请确保回答遵循合适的逻辑关系。
4. 你可以参照<example></example>标记之间的示例设计问答，但请确保完全遵循该示例的格式输出回复。
5.请将示例问题中'（）'内的部分替换为提取的参考文本，并确保不会更改问题的其他部分。

Guidelines

<example>
问题：(玉米生产过程中会出现各种病虫草害的问题，严重影响玉米的生产。通过合理的植保技术手段，可以减少病虫草害的影响，常规的地下害虫和苗期病害以及丝黑穗病都可以用种衣剂解决。玉米苗期低温冷害的问题由于气候多变，春季低温冷害时有发生，严重影响玉米种子正常发芽和出苗，对芽势弱的种子影响更大。解决方案包括适当晚播，选用含精甲霜灵和咯菌腈成份的种衣剂进行种子包衣处理，以及使用"亨达608"对种子进行包衣处理，可直接包衣，农民也可以对已包衣种衣剂的种子进行二次包衣处理。)阅读并理解给定文本，对该文本进行概括总结。
回答：这段文本讨论了玉米生产中遇到的病虫草害问题及其解决方案。强调了通过使用种衣剂和合理的植保技术可以有效减少这些问题的影响。特别提到了春季低温冷害对玉米种子发芽和出苗的不利影响，以及通过晚播和种衣剂处理（包括使用"亨达608"）来提高种子的抗逆性和生长势。
</example>

Example Dialogue

**User:**
请基于给定学术论文，生成*N*个问答对。

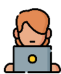

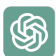

**Example:**
问题1：（玉米防病防虫抗倒伏问题。玉米大小斑病、锈病和虫害是玉米中后期的主要病虫害，发生严重时会造成明显减产。玉米倒伏问题近年来也频繁发生，特别是2020年东北三省遭遇特大台风，造成大面积倒伏，造成机收困难，农民损失很大。解决方案包括国家玉米产业体系研究玉米病虫害防治前移技术，在玉米6-11叶期间喷施的玉米矮化防病增产套餐产品：具体是用"佳节"（30%胺鲜酯·乙烯利)+"享尔"（40%丁香·戊唑醇)+"亨达福"（0.01%芸苔素内酯)+杀虫剂，具有防病、防虫、抗倒伏和增产作用。）阅读并理解给定文本，对该文本进行概括总结。
回答1：这段文本描述了玉米生产中的防病防虫和抗倒伏问题，特别是在遭遇极端天气如台风时的倒伏问题。文中提供了一种解决方案，即使用国家玉米产业体系研究的前移技术，通过在特定生长期喷施一套特定的化学品组合，这些化学品包括胺鲜酯、丁香、戊唑醇和芸苔素内酯，以及杀虫剂，以达到防病、防虫、抗倒伏和增产的效果。
问题2：……
回答2：……

Figure 15: Prompt and few examples for Chinese Summary Q&A.

## Named Entity Recognition (EN)

**Topic Description:**
Instruct the model to understand and extract specific entity information from the provided article.

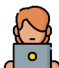

**System:**
You are an expert in the field of study relevant to the academic paper enclosed within the <doc></doc> tags. Please generate a set of question-answer pairs that reflect the key findings and discussions in the paper.

`Role Description`

<doc>
In the recent past, population explosion has augmented urbanization……
</doc>

`Reference Text`

For each question-answer pair, please adhere to the following guidelines:
1. The task specified for the question-answer pair is Named Entity Recognition (NER).
2. Please extract a portion of the paper as the reference text for the task.
3. The entity categories restricted by the task are: Geographic Location, Disease, Pest, Crop Variety.
4. Ensure that the extracted part covers as many entity categories as possible.
5. You may refer to the example provided within the <example></example> tags for designing the question-answer pairs, but ensure that the output strictly follows the format of this example.
6. Please replace the part within '()' in the example question with the extracted reference text, and ensure not to change other parts of the question.

`Guidelines`

<example>
Question: (The \"Golden Dragon 1\" variety has shown resistance to \"rice blast\", a common disease in rice crops. \"Golden Dragon 1\" is suitable for planting in Jiangsu Province. Farmers have reported an increase in the number of rice water weevil in their rice fields.) Read and understand the given text, identify the entities in it and classify them into the following categories: Crop Variety, Geographic Location, and Disease. The reply should be given in dictionary format, where the key is the entity category and the value is a list containing entities belonging to that category.
Answer: {\"Crop Variety\": [\"Golden Dragon 1\"], \"Disease\": [\"rice blast\"], \"Geographic Location\": [\"Jiangsu Province\"]}, \"Pest\": [\"rice water weevil\"]}.
</example>

`Example Dialogue`

**User:**
Please generate *N* question-answer pairs based on the given academic paper.

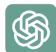

**Example:**
Question 1: (The four maize inbred lines: Agaiti-85 (P1), Golden (P2), Soneri (P3), and Sultan (P4) from F population (99.25% purity) under high temperature (>45°C) and drought stress conditions were selected during 2009-2010 in the University of Agriculture, Faisalabad, Pakistan (31°26'N, 73°06'E).) Read and understand the given text, identify the entities in it and classify them into the following categories: Crop Variety, Geographic Location. The reply should be given in dictionary format, where the key is the entity category and the value is a list containing entities belonging to that category.
Answer 1: {"Crop Variety": ["Agaiti-85", "Golden", "Soneri", "Sultan"], "Geographic Location": ["Faisalabad, Pakistan"]}.
Question 2: (The intensities of crop damage caused by thrips, armyworm, fruit fly, bacterial wilt (Ralstonia solanacearum), and gemini virus (pepper yellow leaf curl virus) infections were then determined.) Read and understand the given text, identify the entities in it and classify them into the following categories: Disease, Pest. The reply should be given in dictionary format, where the key is the entity category and the value is a list containing entities belonging to that category.
Answer 2: {"Disease": ["bacterial wilt", "gemini virus"], "Pest": ["thrips", "armyworm", "fruit fly"]}.
Question 3: ……
Answer 3: ……

Figure 16: Prompt and few examples for English Named Entity Recognition Q&A.

**Named Entity Recognition (ZH)**

**Topic Description:**
Instruct the model to understand and extract specific entity information from the provided article.

**System:**
你是一名育种专家，同时是一位农业科学和生物科学的教授，请基于在
<doc></doc>标记之间的这篇学术论文，生成命名实体识别问答对。

Role Description

<doc>
玉米营养诊断是在玉米生育进程中的某些时期检测植株的营养状况是否符合……
</doc>

Reference Text

对于每一个问答对，你需要注意以下几点：
1.问答所指定的任务为命名实体识别（NER）。
2.请截取论文的一部分作为任务的参考文本。
3.任务限制的实体类别为：地理位置，病虫害，作物品种。
4.请确保所截取部分涵盖尽可能多的实体类别。
5.你可以参照<example></example>标记之间的示例设计问答，但请确保完全遵循
该示例的格式输出回复。
6.请将示例问题部分中'（）'之间的部分替换为所截取的参考文本，确保不要更改
问题的其他部分。

Guidelines

<example>
问题：（"金龙1号"品种已经显示出对"稻瘟病"的抗性，这是水稻作物中的一种常
见疾病。"金龙1号"适合在江苏南部种植。然而，农民们报告说，在他们的稻田里，
一种害虫"稻象甲"的数量有所增加。）阅读并理解给定文本，识别其中的实体并
将其分类为以下类别：作物品种、地理位置和疾病。回复应该以字典格式给出，
其中键为实体类别，值为含有所属该类别实体的列表。
回答：{"作物品种": ["金龙1号"], "疾病": ["稻瘟病"], "地理位置": ["江苏南部"]}
</example>

Example
Dialogue

**User:**
请基于给定学术论文，生成*N*个问答对。

**Example:**
问题1：（"2023年河北省农科院植保所郭宁等统计在2018-2020年黄淮海区427个玉米国审品种中，
南方锈病的抗性以高感和感病为主，未发现高抗品种，抗性品种和中抗品种的数量也较少。对
2020年黄淮海主栽品种也进行了抗性分析。"）阅读并理解给定文本，识别其中的实体并将其分
类为以下类别：作物品种、地理位置和疾病。回复应该以字典格式给出，其中键为实体类别，值
为含有所属该类别实体的列表。
回答1：{"作物品种": ["玉米"], "病虫害": ["南方锈病"], "地理位置": ["河北省", "黄淮海区"]}。
问题2：（美国转基因玉米面积约占全国玉米面积的93%，主要以抗虫和抗除草剂叠加性状为主。
由于转基因玉米品种的推广，大幅减少了虫害（玉米螟、黏虫、根虫等）对产量的影响，相当于
增产10~15%。而我国玉米基本都是常规玉米（除近年来一些地区非法种植了小部分转基因玉米
外），生产上虫害频发（玉米螟、黏虫、棉铃虫、桃蛀螟、草地贪夜蛾等），给玉米的产量造成
了很大损失，同时也严重影响了玉米的商品品质。）阅读并理解给定文本，识别其中的实体并将
其分类为以下类别：作物品种、地理位置和疾病。回复应该以字典格式给出，其中键为实体类别，
值为含有所属该类别实体的列表。
回答2：{"作物品种": ["转基因玉米", "常规玉米"], "病虫害": ["玉米螟", "黏虫", "根虫", "棉铃虫",
"桃蛀螟", "草地贪夜蛾"]}。
问题3：……
回答3：……

Figure 17: Prompt and few examples for Chinese Named Entity Recognition Q&A.

**Event Extraction (EN)**

**Topic Description:**
Instruct the model to identify the event triggers and extract relevant arguments associated with those events from the provided article.

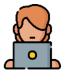

**System:**
You are an expert in the field of study relevant to the academic paper enclosed within the <doc></doc> tags. Please generate a set of question-answer pairs that reflect the key findings and discussions in the paper.

**Role Description**

<doc>
In the recent past, population explosion has augmented urbanization……
</doc>

**Reference Text**

For each question-answer pair, please adhere to the following guidelines:
1. The task specified for the question-answer pair is Event Extraction.
2. Please extract a portion of the paper as the reference text for the task.
3. The event categories restricted by the task are: No Event, Pest Impact, New Variety Release or Breeding, Climate or Weather Event, Planting or Production Trial, Policy Release, Yield Record, Variety Mutation.
4. Ensure that the extracted part belongs to only one event category and highly matches that event category.
5. You may refer to the example provided within the <example></example> tags for designing the question-answer pairs, but ensure that the output strictly follows the format of this example.
6. Please replace the part within '()' in the example question with the extracted reference text, and ensure not to change other parts of the question.

**Guidelines**

<example>
Question: (Last week, the \"Golden Dragon 1\" variety was officially released in the Philippines. This variety was developed by the International Rice Research Institute (IRRI) and was biofortified to combat Vitamin A deficiency. Farmers, scientists, and government officials attended the release event. It was announced at the event that the \"Golden Dragon 1\" seeds will be distributed to local farmers in the next planting season.) Identify the main event type and provide a summary of the event from the given text. The result should be presented in a dictionary format where the key is the event type and the value is a summary of the event.
Answer: {\"New Variety Release or Breeding\":[\"The 'Golden Dragon 1' variety, developed by the International Rice Research Institute (IRRI) to combat Vitamin A deficiency, was officially released in the Philippines. The seeds will be distributed to local farmers in the next planting season.\"]}
</example>

**Example Dialogue**

**User:**
Please generate *N* question-answer pairs based on the given academic paper.

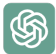

**Example:**
Question 1: (The seedlings were grown for two weeks in a glasshouse under a maximum natural daylight irradiance of 1200-1600μmol m² s, 21/27-13/16°C day/night temperature, 14/10 h light/dark cycle, and relative humidity between 65 and 70 % in pots filled with a mixture of vermiculite and Trio-hum tray substrate, being irrigated with Hoagland nutrient solution.) Identify the main event type and provide a summary of the event from the given text. The result should be presented in a dictionary format where the key is the event type and the value is a summary of the event.
Answer 1: {"Planting or Production Trial":["Wheat and maize seedlings were cultivated in a glasshouse with controlled environmental conditions including light, temperature, and humidity for two weeks to ensure optimal growth before undergoing chilling stress trials."]}
Question 2: ……
Answer 2: ……

Figure 18: Prompt and few examples for English Event Extraction.

**Event Extraction (ZH)**

<div style="border:1px solid">

**Topic Description:**
Instruct the model to identify the event triggers and extract relevant arguments associated with those events from the provided article.

</div>

**System:**
你是一名育种专家，同时是一位农业科学和生物科学的教授，请基于在 <doc></doc>标记之间的这篇学术论文，生成事件提取问答对。 — **Role Description**

<doc>
玉米营养诊断是在玉米生育进程中的某些时期检测植株的营养状况是否符合…… </doc> — **Reference Text**

对于每一个问答对，你需要注意以下几点：
1.问答所指定的任务为事件提取（Event Extraction）。
2.请截取论文的一部分作为任务的参考文本。
3.任务限制的事件类别为：无事件，病虫害影响，新品种发布或选育，气候或天气事件，种植或生产试验，政策发布，产量新纪录，品种变异。
4.请确保所截取部分仅属于一个事件类别，且与该事件类别高度匹配。
5.你可以参照<example></example>标记之间的示例设计问答，但请确保完全遵循该示例的格式输出回复。
6.请将示例问题部分中'（）'之间的部分替换为所截取的参考文本，确保不要更改问题的其他部分。 — **Guidelines**

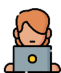

<example>
问题：（上周，"金龙1号"品种在菲律宾正式发布。该品种是由国际水稻研究所（IRRI）开发的，经过生物强化以对抗维生素A缺乏症。农民、科学家和政府官员参加了此次发布活动。活动中宣布，"金龙1号"种子将在下一个种植季节分发给当地农民。）阅读并理解给定文本，识别其中的事件并将其分类。回复应该以字典格式给出，其中键为事件类别，值为含有事件高度概括的列表。
回答：{"新品种发布或选育":["国际水稻研究所（IRRI）为对抗维生素A缺乏症而开发的"黄金大米"品种在菲律宾正式发布。种子将在下一个种植季节分发给当地农民。"]}
</example> — **Example Dialogue**

**User:**
请基于给定学术论文，生成*N*个问答对。

<div style="border:1px solid">

**Example:**
问题1：（玉米杂交种丹农玉788是丹东农业科学院2017年以F5866 为母本、外引系沈1554为父本杂交选育而成。2017 年参加品种比较试验，2018-2019年参加联合比较试验，试验名称为丹3202，2020-2021 年参加中科玉科企联合体中晚熟组区域试验，2021年参加中科玉科企联合体中熟组生产试验。2022 年通过国家品种审定委员会审定(审定编号：国审玉20220202)，定名为丹农玉788。）阅读并理解给定文本，识别其中的事件并将其分类。回复应该以字典格式给出，其中键为事件类别，值为含有事件高度概括的列表。
回答1：{"新品种发布或选育":["丹东农业科学院选育的玉米杂交种丹农玉788，经过多年的试验和审定，最终在2022年被正式命名并发布。"]}
问题2：……
回答2：……

</div>

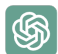

Figure 19: Prompt and few examples for Chinese Event Extraction.

**Open-book Q&A (EN)**

**Topic Description:**
Instruct the model to return professional knowledge Q&A. There are external reference materials provided in the produced question.

**System:**
You are an expert in the field of study relevant to the academic paper enclosed within the <doc></doc> tags. Please generate a set of question-answer pairs that reflect the key findings and discussions in the paper.

Role Description

<doc>
In the recent past, population explosion has augmented urbanization……
</doc>

Reference Text

For each question-answer pair, please adhere to the following guidelines:
1. The task specified for the question-answer pair is Direct Question Answering.
2. Please extract a portion of the paper (at least 300 words) as the reference text for the task.
3. The question part of the question-answer pair should include the reference text and the generated question, and the answer part is the matching answer.
4. If the paper provides relevant information, ensure to include the following aspects: Stress Resistance, Pests and Diseases, Cultivation Techniques, Breeding, Characteristics, Suitable Planting Area, etc.
5. You can design the question-answer pair according to the example within the <example></example> tags, but ensure the output strictly follows the format of the example.
6. Please replace the part within '()' in the example question with the extracted reference text, and the part within '[]' with the generated question.

Guidelines

<example>
Question: (\"Golden Dragon 1\" is a type of rice produced through genetic engineering to biosynthesize beta-carotene, a precursor of vitamin A, in the edible parts of rice. It is intended to produce a fortified food to be grown and consumed in areas with a shortage of dietary vitamin A.) [What is the main purpose of developing Golden Rice?]
Answer: The main purpose of developing \"Golden Dragon 1\" is to produce a fortified food to be grown and consumed in areas with dietary Vitamin A deficiency.
</example>

Example Dialogue

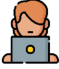

**User:**
Please generate *N* question-answer pairs based on the given academic paper.

---

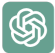

**Example:**
Question 1: ("The diseases detected in the experiment were southern rust, maize bushy stunt, corn stunt and corn smut. Southern rust was the prevailing disease, with the highest incidence (76.94%). Caused by the fungus Puccinia polysora Undrew, this disease is currently one of the most important diseases of maize in Brazil and is considered the most aggressive rusts that occur on this crop.") [What was the most prevalent disease found in the maize crops during the study?]
Answer 1: The most prevalent disease found in the maize crops during the study was southern rust, caused by the fungus Puccinia polysora Undrew.
Question 2: ("The highest prevalence and severity of southern rust in maize was found in crops using the high technology system tested. Statistical analyses detected that rust severity increased along with the technology level used for maize cropping.") [How did the technology level used in maize cropping affect the severity of southern rust?]
Answer 2: The severity of southern rust increased with the technology level used in maize cropping, with the highest prevalence and severity found in crops using the high technology system.
Question 3: ……
Answer 3: ……

Figure 20: Prompt and few examples for English Open-book Q&A.

**Open-book Q&A (ZH)**

**Topic Description:**
Instruct the model to return professional knowledge Q&A. There are external reference materials provided in the produced question.

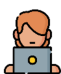

**System:**
你是一名育种专家，同时是一位农业科学和生物科学的教授，请基于在
<doc></doc>标记之间的这篇学术论文，生成问答对。 | Role Description

<doc>
玉米营养诊断是在玉米生育进程中的某些时期检测植株的营养状况是否符合…… | Reference Text
</doc>

对于每一个问答对，你需要注意以下几点：
1.问答所指定的任务为直接问答（Direct Question Answering）。
2.请截取论文的一部分（至少200字）作为任务的参考文本。
3.问答对的问题部分包括参考文本和生成的问题，回答部分为匹配的回答。
4.如果论文中有相关信息，请包含涉及选种过程；特征特性；栽培技术要点；适宜种植区域；病虫害防治；抗逆性等方面的问题。
5.你可以参照<example></example>标记之间的示例设计问答，但请确保完全遵循该示例的格式输出回复。
6.请将示例问题部分中'（）'之间的部分替换为所截取的参考文本，'[]'之间的部分替换为生成的问题。 | Guidelines

<example>
问题：（"金龙1号"是一种水稻，通过基因工程在大米的可食用部分生物合成维生素a的前体β-胡萝卜素。它旨在生产一种强化食品，在膳食维生素a短缺的地区种植和食用。）[开发黄金大米的主要目的是什么？]
回答：开发"金龙1号"的主要目的是生产一种强化食品，在膳食维生素a短缺的地区种植和食用。
</example> | Example Dialogue

**User:**
请基于给定学术论文，生成*N*个问答对。

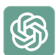

**Example:**
问题1：（玉米种子质量和成本是良种推广的重要因素，也体现品种价值。长期以来，玉米育种家在制定玉米育种目标时，多重视杂种一代的配合力，特别是杂种一代的产量、质量和抗性，轻视玉米制种产量、质量及其技术，从而影响了品种的推广。）[在玉米育种中，为什么制种产量和质量同样重要？]
回答1：在玉米育种中，制种产量和质量同样重要，因为它们是良种推广的重要因素，体现了品种的价值。如果忽视这些方面，将影响品种的推广。
问题2：（甘肃、新疆、宁夏由于地处我国内陆，光照充足，昼夜温差大，气候干燥，病虫害轻，是我国乃至世界最佳制种基地。针对上述存在问题，通过多年实践，探索创新了一些制种技术。）[为什么甘肃、新疆、宁夏被认为是最佳的玉米制种基地？]
回答2：甘肃、新疆、宁夏被认为是最佳的玉米制种基地，因为这些地区地处内陆，具有充足的光照、大的昼夜温差和干燥的气候，同时病虫害较轻，这些条件都非常适合玉米制种。
问题3：……
回答3：……

Figure 21: Prompt and few examples for Chinese Open-book Q&A.

### J.3    Multi-turn Q&A

We design a multi-agent setting for the generation of multi-turn Q&As, where an assistant agent and a user agent are defined with different roles. Meanwhile, for each type of crop, we define six tasks under three scenarios (problem-solving, personalized recommendation, and knowledge interpretation). Thus, prompts are determined by the role of the agent and specific tasks. Since our proposed multi-turn Q&A prompts are task-dependent, we only present the prompt of the pest control task as an example, illustrated in Figure 22 and 23, with both an English version and a Chinese version.

**Problem Solving—Pest Control.**    The dialogue involves identifying and providing solutions for pest problems in crops based on the farmer's descriptions and queries. The following aspects of the pest infestation scenarios are considered: infestation identification, infestation control, post-infestation recovery, *etc*. The example is shown in Figure 24.

**Problem Solving—Nutrient Supplementation.**    The dialogue involves identifying and providing solutions for nutrient deficiency problems in crops based on the farmer's descriptions and queries. The following aspects of the nutrient deficiency scenarios are considered: deficiency analysis, soil testing, nutrient balancing, *etc*. The example is shown in Figure 25.

**Problem Solving—Disease Containment.**    The dialogue involves identifying and providing solutions for disease problems in crops based on the farmer's descriptions and queries. The following aspects of the disease outbreak scenarios are considered: disease identification, emergency treatment, post-outbreak recovery, *etc*. The example is shown in Figure 26.

**Personalized Recommendation—Crop Variety Selection.**    The dialogue involves recommending crop varieties that are best suited to the farmer's specific conditions (such as soil types, climate, and disease resistance needs) and goals (such as yield, and quality). The following aspects of the crop variety selection scenarios are considered: pest and disease resistance, tolerance to abiotic stresses, nutrient requirement analysis, *etc*. The example is shown in Figure 27.

**Personalized Recommendation—Resource Management.**    The dialogue offers tailored advice on efficient resource management, which might include the optimal use of water, fertilizers, and labor, to maximize crop yield and profitability while minimizing environmental impact. The following aspects of the resource management scenarios are considered: precision agriculture, sustainable practices, climate-resilient practices, *etc*. The example is shown in Figure 28.

**Knowledge Interpretation—Research Interpretation.**    The dialogue should offer a clear interpretation of the specific research findings to help the farmer stay updated with discoveries in the crop science domain, regardless of when the research was conducted. The following research interpretation scenarios are considered: understanding the research methods, interpreting the results, discussing the implications, *etc*. The example is shown in Figure 29.

### J.4    Benchmark

Multiple-choice Q&As are extracted from the data source using prompts shown in Figure 30 and 31, with both English and Chinese versions.

## K    Training Details

Our fine-tuning work was performed based on the LLaMA-Factory framework [70]. To mitigate the risk of overfitting during the tuning process[65, 15], we augmented our training data with the COIG-CQIA dataset[5]. We used the LoRA[32] method to train all three models, with rank set to 8, alpha set to 16, and learning rate set to 5e-5. Additionally, we deployed Deepspeed ZeRO-2[49, 48] to improve memory efficiency and set the total train batch size to 320, and epoch to 4.

All fine-tuning was performed with 16 NVIDIA A100-40G GPUs. During inference, we set a maximum input length to 4096 tokens, and an output length to 1024 tokens for weights-access models. For API-access models, we set the temperature to 0, and the top $p$ to 1.

**Multi-turn Q&A (EN)**

**Assistant System Prompt:**
You are a crop pestologist (assistant). I am a farmer (user), who approaches you with concerns about my crop. Your role is to engage in a dialogue with me, ask insightful questions, and provide information to help identify the problem and suggest potential solutions based on your expertise.

<doc>
RAG contents……
</doc>

Please adhere to the following guidelines:
1. As a Crop Pestologist, you have access to a broad range of professional domain knowledge. Use this knowledge to inform your responses and provide accurate, helpful advice.
2. For each round, you have access to a reference text that is highly relevant to the conversation content. Use this context to inform your responses where relevant. The reference text for this round is enclosed within the <doc></doc> tags.
3. Ensure that you only consider the following knowledge-intensive problem-solving scenario: Pest Infestation. The dialogue involves identifying and providing solutions for pest problems in crops based on the farmer's descriptions and queries.
4. You must maintain your role as the crop pestologist (assistant) throughout the conversation. Never flip roles.
5. You must provide your responses one at a time, ensuring they are non-empty.
6. Ensure to terminate the conversation timely when I conclude the conversation with 'Thank you, I got the information I needed' by responding with 'You are welcome'.
7. Ensure that your output strictly follows the format of the example enclosed within the <example></example> tags, and replace the content enclosed within the [] tags with your response.

<example>
Crop Pestologist (Assistant): []
</example>

**User System Prompt:**
You are a farmer (user) who is concerned about the health of your crops. You approach me, a crop pestologist (assistant) with your concerns. Your role is to engage in a dialogue with me, describe the symptoms you observed, ask questions, and seek advice.

<doc>
RAG context……
</doc>

Please adhere to the following guidelines:
1. As a farmer, you have hands-on experience with your crops. Use this knowledge to provide detailed descriptions or queries.
2. For each round, you have access to a reference text that is highly relevant to the conversation. Use this context to inform your descriptions and queries where relevant. The reference text for this round is closed within the <doc></doc> tags.
3. You are only part of the following knowledge-intensive problem-solving scenario: Pest Infestation. The dialogue involves describing your crop symptoms and asking for solutions to potential pest problems.
4. You must maintain your role as the farmer (user) throughout the conversation. Never flip roles.
5. You must provide your descriptions or queries one at a time, ensuring they are non-empty.
6. Ensure to terminate the conversation timely when you have obtained the information you need by saying 'Thank you, I got the information I needed'.
7. Ensure that your output strictly follows the format of the example enclosed within the <example></example> tags, and replace the content enclosed within the [] tags with your response.

<example>
Farmer (User): []
</example>

Figure 22: English Prompt for multi-turn Q&A. We take the pest control task as an example to show our multi-agent system prompts.

**Multi-turn Q&A (ZH)**

**Assistant System Prompt:**
你是一位作物病虫害专家（助理）。我是一位农民（用户），我因对作物健康状况感到担忧而向你寻求帮助。你的任务是与我进行对话，根据你的专业知识提供信息，帮助识别问题并提出可能的解决方案。

<doc>
RAG contents……
</doc>

请遵循以下准则：
1. 你可以获取广泛的专业领域知识。使用这些知识来完善你的回答从而提供准确、有用的建议。
2. 对于每一轮对话，你都可以访问一段与对话内容高度相关的参考文本。请使用此上下文来补充你的回复。本轮的参考文本包含在<doc></doc>标记之间。
3. 请确保你只考虑以下知识密集型问题解决场景：农作物害虫侵害。这种场景中的对话涉及根据农民的描述和询问，识别农作物害虫问题并提供适配的解决方案。
4. 你必须在整个对话中保持你作为作物病虫害专家（助理）的角色。不要切换为其他角色。
5. 你必须一次仅提供一个回复，且该回复应当非空。
6. 当我以"谢谢，我得到了我需要的信息"请求结束对话时，确保及时通过回复"不客气"来终止对话。
7. 请确保完全遵循<example></example>标记之间示例的格式输出回复，将【】标记之间的内容替换为你的回复。

<example>
作物病虫害专家（助理）：【】
</example>

**User System Prompt:**
你是一位农民（用户），你对作物的健康状况感到担忧。我是一位作物病虫害专家（助理），你向我寻求帮助。你的任务是与我进行对话，描述你观察到的症状，提出问题，并根据我的专业知识寻求可能的解决方案。

<doc>
RAG context……
</doc>

请遵循以下准则：
1. 你拥有自己作物的第一手经验。请利用这些知识给出详细的描述或查询。
2. 对于每一轮对话，你都可以访问一段与对话内容高度相关的参考文本。请使用此上下文来补充你的描述或查询。本轮的参考文本包含在<doc></doc>标记之间。
3. 你仅参与以下知识密集型问题解决场景：农作物害虫侵害。这种场景中的对话涉及描述作物的症状并寻求潜在害虫问题的解决方案。
4. 你必须在整个对话中保持你作为农民（用户）的角色。不要切换为其他角色。
5. 你必须一次仅提供一个描述或查询，且该描述或查询应当非空。
6. 当你已经获得所需的信息时，确保通过回复"谢谢，我得到了我需要的信息"来及时终止对话。
7. 请确保完全遵循<example></example>标记之间示例的格式输出回复，将【】标记之间的内容替换为你的回复。

</example>
农民（用户）：【】
</example>

Figure 23: Chinese Prompt for multi-turn Q&A. We take the pest control task as an example to show our multi-agent system prompts.

**Problem Solving-Pest Control**

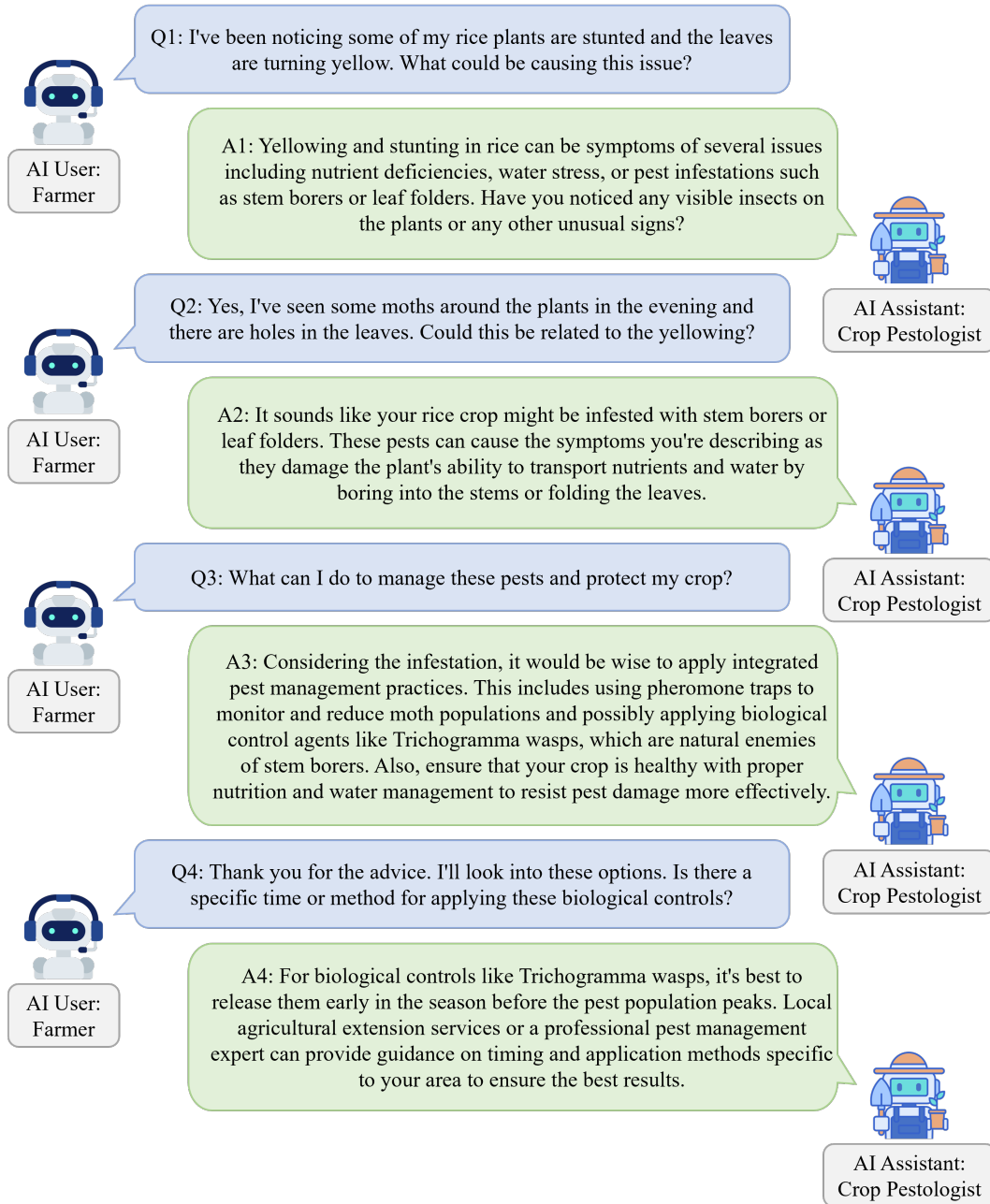

Figure 24: The example for the pest control task of problem-solving scenario.

**Problem Solving-Nutrient Supplementation**

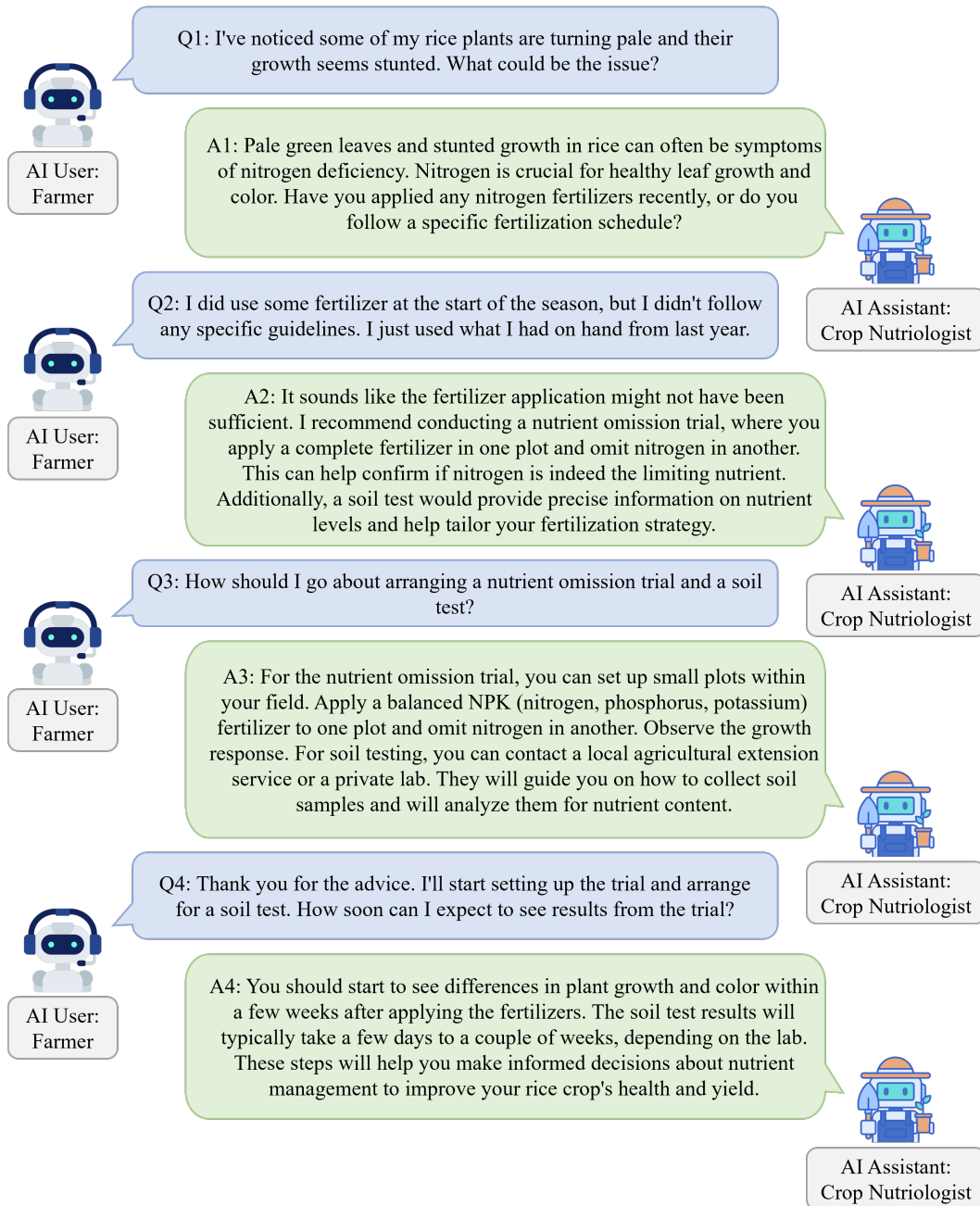

Figure 25: The example for the nutrient supplementation task of problem-solving scenario.

**Problem Solving-Disease Containment**

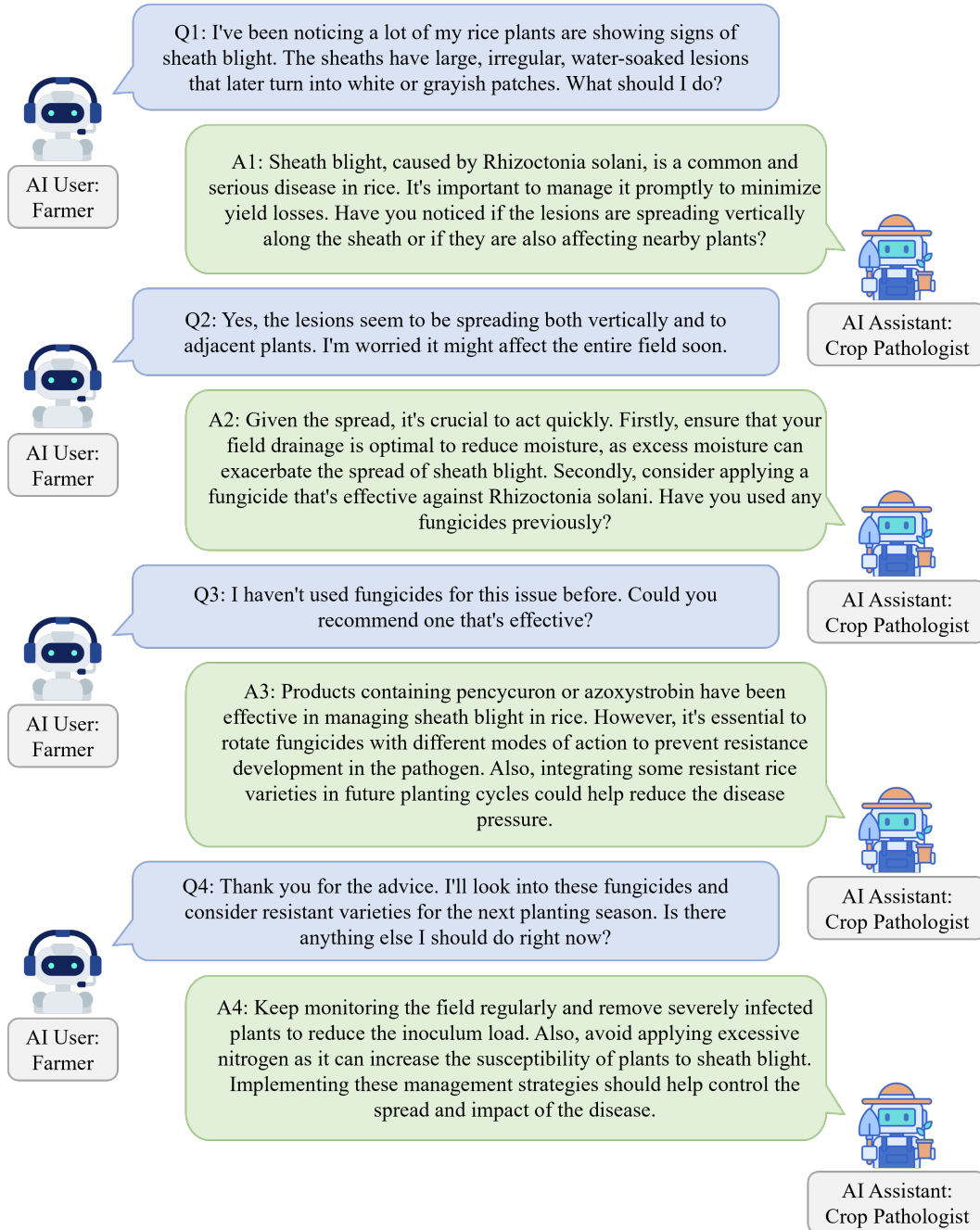

Figure 26: The example for the disease containment task of problem-solving scenario.

**Problem Solving-Crop Variety Selection**

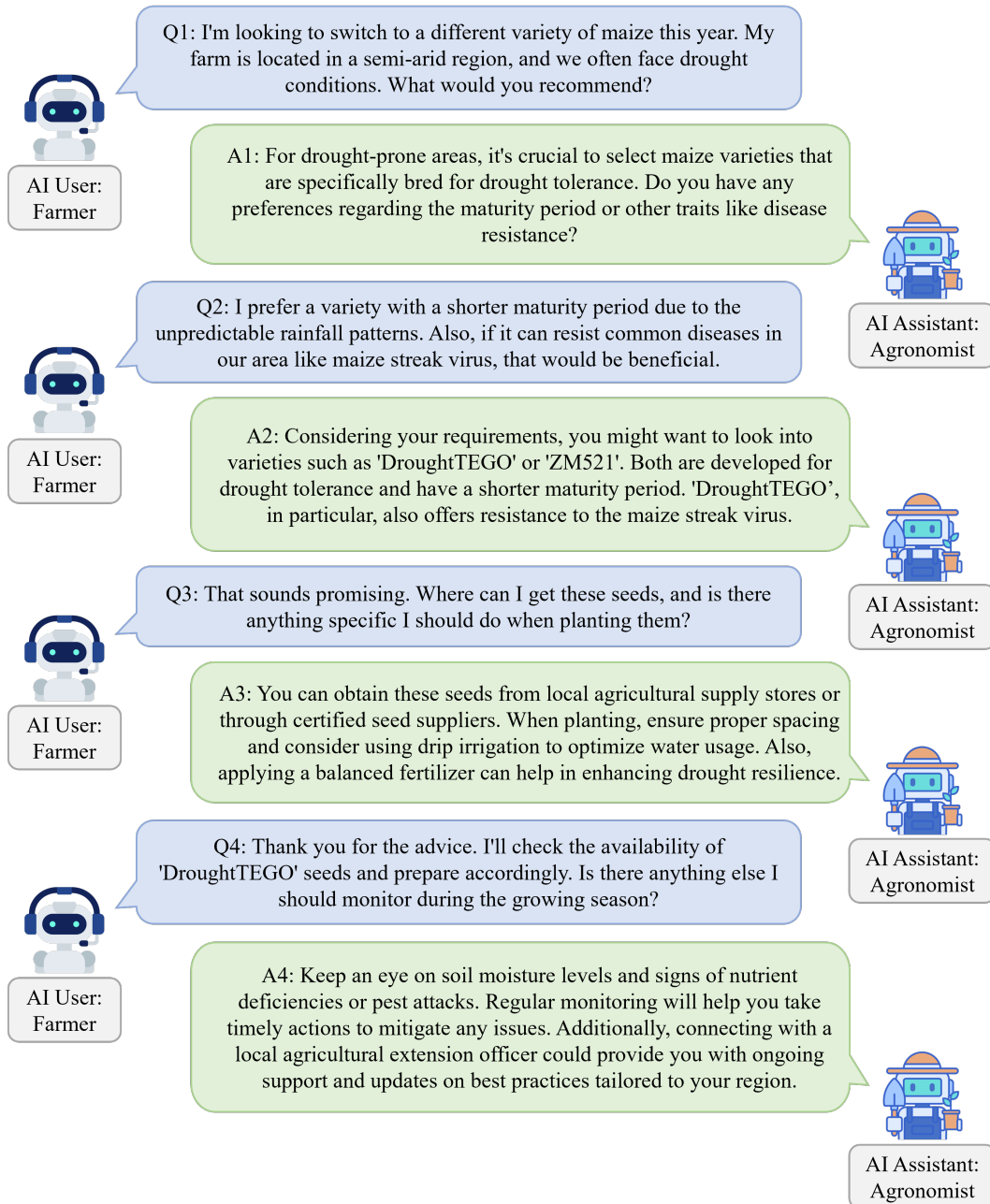

Q1: I'm looking to switch to a different variety of maize this year. My farm is located in a semi-arid region, and we often face drought conditions. What would you recommend?

A1: For drought-prone areas, it's crucial to select maize varieties that are specifically bred for drought tolerance. Do you have any preferences regarding the maturity period or other traits like disease resistance?

Q2: I prefer a variety with a shorter maturity period due to the unpredictable rainfall patterns. Also, if it can resist common diseases in our area like maize streak virus, that would be beneficial.

A2: Considering your requirements, you might want to look into varieties such as 'DroughtTEGO' or 'ZM521'. Both are developed for drought tolerance and have a shorter maturity period. 'DroughtTEGO', in particular, also offers resistance to the maize streak virus.

Q3: That sounds promising. Where can I get these seeds, and is there anything specific I should do when planting them?

A3: You can obtain these seeds from local agricultural supply stores or through certified seed suppliers. When planting, ensure proper spacing and consider using drip irrigation to optimize water usage. Also, applying a balanced fertilizer can help in enhancing drought resilience.

Q4: Thank you for the advice. I'll check the availability of 'DroughtTEGO' seeds and prepare accordingly. Is there anything else I should monitor during the growing season?

A4: Keep an eye on soil moisture levels and signs of nutrient deficiencies or pest attacks. Regular monitoring will help you take timely actions to mitigate any issues. Additionally, connecting with a local agricultural extension officer could provide you with ongoing support and updates on best practices tailored to your region.

AI User: Farmer

AI Assistant: Agronomist

Figure 27: The example for the crop variety selection task of personalized recommendation scenario.

**Problem Solving-Resource Management**

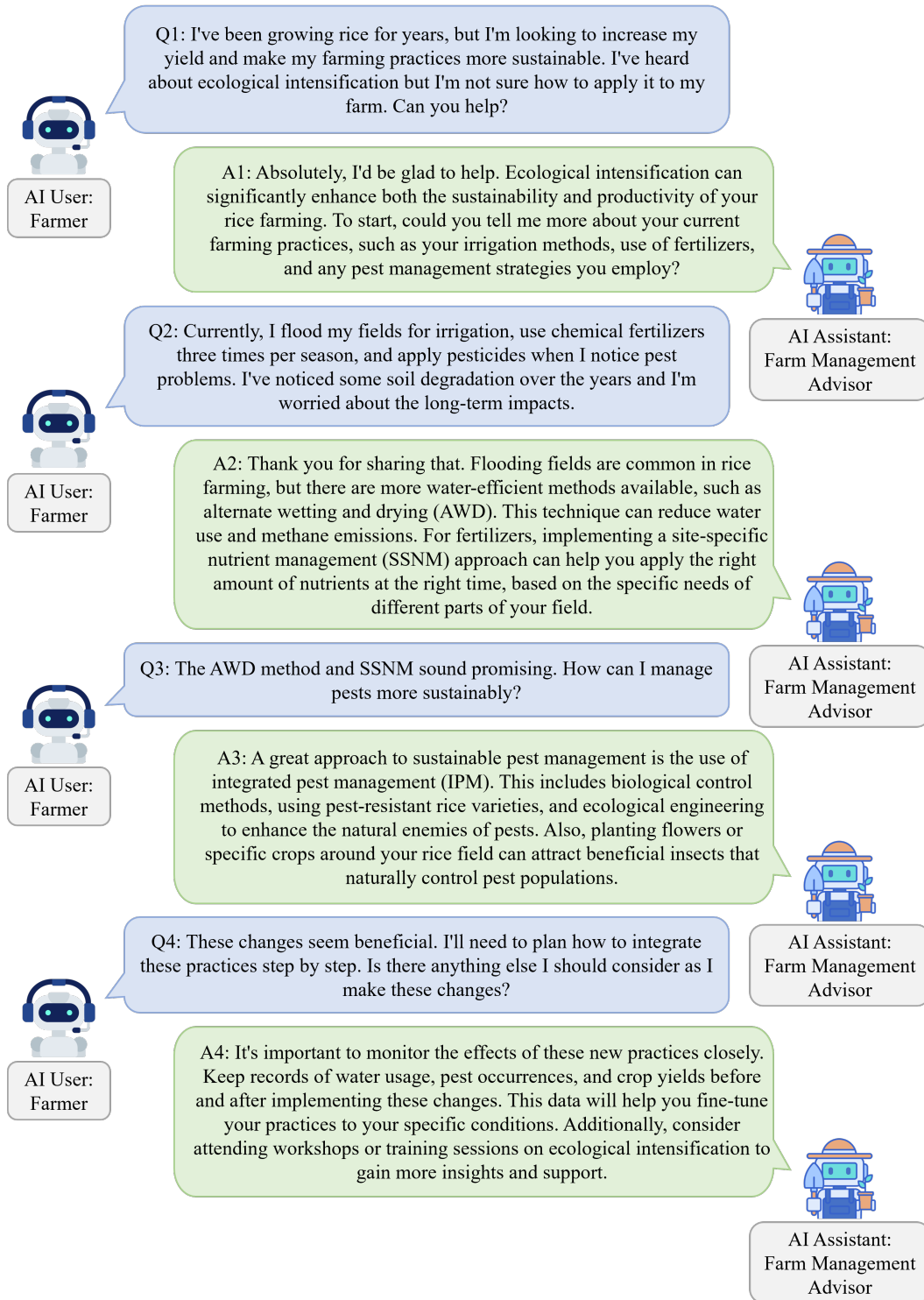

Figure 28: The example for the resource management task of personalized scenario.

**Problem Solving-Research Interpretation**

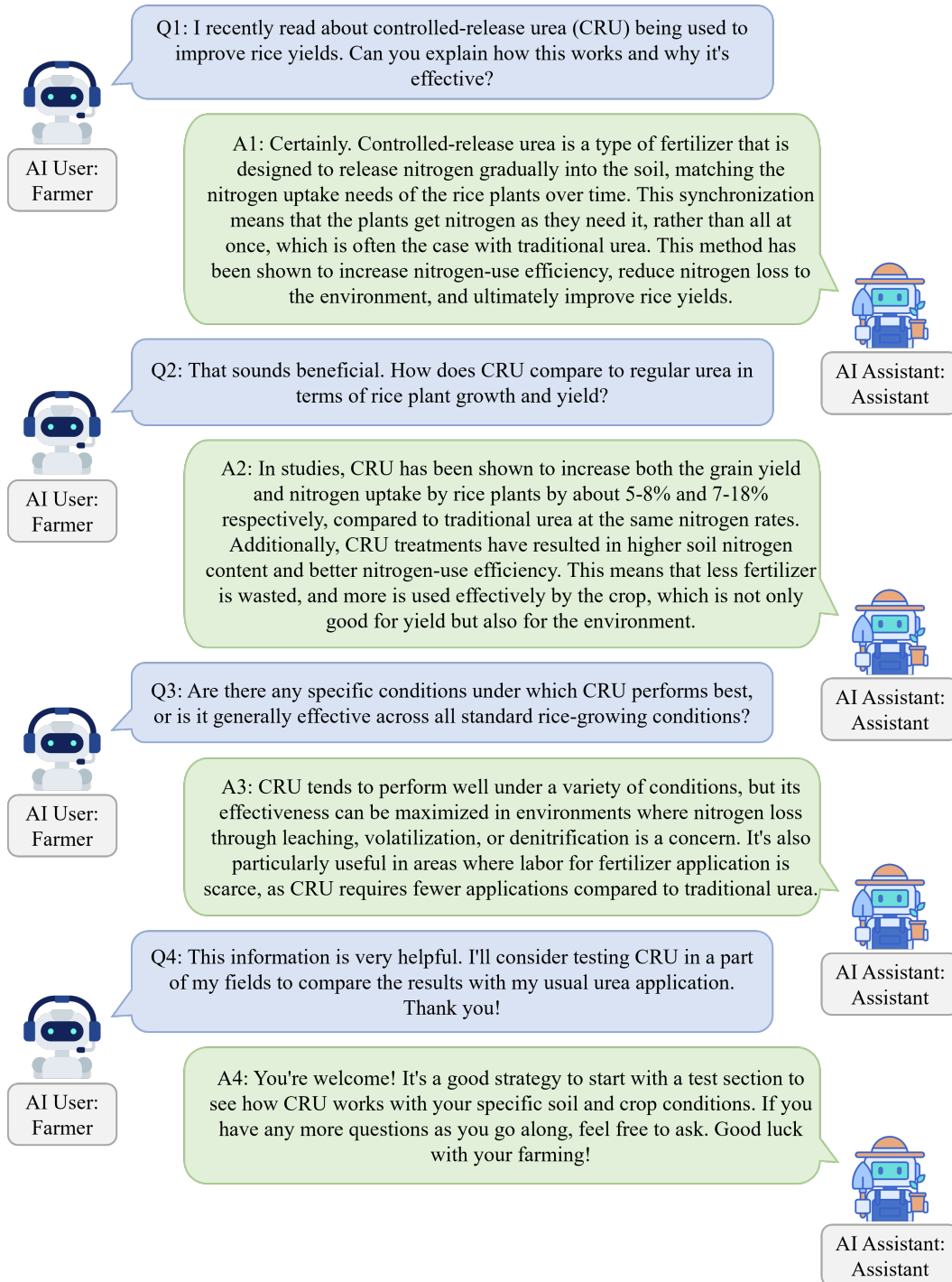

Q1: I recently read about controlled-release urea (CRU) being used to improve rice yields. Can you explain how this works and why it's effective?

A1: Certainly. Controlled-release urea is a type of fertilizer that is designed to release nitrogen gradually into the soil, matching the nitrogen uptake needs of the rice plants over time. This synchronization means that the plants get nitrogen as they need it, rather than all at once, which is often the case with traditional urea. This method has been shown to increase nitrogen-use efficiency, reduce nitrogen loss to the environment, and ultimately improve rice yields.

Q2: That sounds beneficial. How does CRU compare to regular urea in terms of rice plant growth and yield?

A2: In studies, CRU has been shown to increase both the grain yield and nitrogen uptake by rice plants by about 5-8% and 7-18% respectively, compared to traditional urea at the same nitrogen rates. Additionally, CRU treatments have resulted in higher soil nitrogen content and better nitrogen-use efficiency. This means that less fertilizer is wasted, and more is used effectively by the crop, which is not only good for yield but also for the environment.

Q3: Are there any specific conditions under which CRU performs best, or is it generally effective across all standard rice-growing conditions?

A3: CRU tends to perform well under a variety of conditions, but its effectiveness can be maximized in environments where nitrogen loss through leaching, volatilization, or denitrification is a concern. It's also particularly useful in areas where labor for fertilizer application is scarce, as CRU requires fewer applications compared to traditional urea.

Q4: This information is very helpful. I'll consider testing CRU in a part of my fields to compare the results with my usual urea application. Thank you!

A4: You're welcome! It's a good strategy to start with a test section to see how CRU works with your specific soil and crop conditions. If you have any more questions as you go along, feel free to ask. Good luck with your farming!

Figure 29: The example for the research interpretation task of knowledge interpretation scenario.

**Multiple-Choice (EN)**

**Topic Description:**
Instruct the model to return multiple-choice from the provided article. There are no external reference materials provided in the produced question.

**System:**
You are a breeding expert, as well as a professor of agricultural science and biology. Please generate a set of question-answer pairs for this academic paper between the <doc></doc> tags.

Role Description

<doc>
In the recent past, population explosion has augmented urbanization……
</doc>

Reference Text

For each QA pair, you need to consider the following:
1.The task specified for the question-answer pair is Multiple-Choice.
2.The question part of the question-answer pair should include the generated question. The answer part consists of 4 options and the correct answer.
3.Please ensure that each option has significant differences but similar lengths.
4.Please ensure that the production of questions and answers is related to the knowledge of crop cultivation or the properties of crops.
5.You can refer to the example between the <example></example> tags for the design of QA, but please ensure to strictly follow the format of that example.
6.Replace the part within '[]' in the example answer with 4 options, and the part within '{}' with the correct answer.

Guidelines

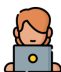

<example>
Question: What was the experimental design used to study the decomposition of Bt and non-Bt rice residues?
Answer: [Options: A. The experimental design was a completely randomized design with two types of rice residues, KMD (Bt) and Xiushui 11 (non-Bt), cultivated in six plots over two successive years using conventional intensive rice cropping methods.
B. The experimental design was a stratified random design with two types of rice residues, KMD (Bt) and Xiushui 11 (non-Bt), cultivated in four plots over three successive years using conventional intensive rice cropping methods.
C. The experimental design was a block design with three types of rice residues, including KMD (Bt) and Xiushui 11 (non-Bt), cultivated in eight plots over two successive years using organic rice cropping methods.
D. The experimental design was a factorial design with two types of rice residues, KMD (Bt) and Xiushui 11 (non-Bt), cultivated in ten plots over one year using conventional intensive rice cropping methods.]
{Correct answer: A.}
</example>

Example Dialogue

**User:**
Please generate *N* question-answer pairs based on the given academic paper.

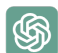

**Example:**
Question 1: How was the herbicide resistance gene introduced into the restorer line Bar68-1?
Answer 1: [Options: A. By crossbreeding Bar68-1 with naturally resistant wild rice strains.
B. Through traditional selective breeding techniques focusing on natural mutations.
C. By particle bombardment using gold particles coated with plasmid DNA containing the bar gene.
D. By chemical induction of mutations in the Bar68-1 line followed by selective breeding.]
{Correct answer: C.}
Question 2：……
Answer 2：……

Figure 30: Prompt and few examples for English Multi-Choice Q&A.

**Multiple-Choice (ZH)**

**System:**
你是一名育种专家，同时是一位农业科学和生物科学的教授，请基于在<doc></doc>标记之间的这篇学术论文，生成问答对。

Role Description

<doc>
玉米营养诊断是在玉米生育进程中的某些时期检测植株的营养状况是否符合……
</doc>

Reference Text

对于每一个问答对，你需要注意以下几点：
1.问答所指定的任务为多项选择题（Multiple-Choice）。
2.问答对的问题部分为生成的问题，回答部分为4个选项和正确答案。
3.请确保每个选项的内容差异化很大，但是长度相近。
4.请确保生产的问题和答案与农作物种植知识或农作物性质相关。
5.你可以参照<example></example>标记之间的示例设计问答，但请确保完全遵循该示例的格式输出回复。
6.请将示例回答部分中'[]'之间的部分替换为4个选项，'{}'之间的部分替换为正确答案。

Guidelines

<example>
问题："白梗一号"水稻的主要农艺特性有哪些？
回答：[选项：A、"白梗一号"水稻的主要农艺特性包括中早熟品种，生育期131天，需活动积温2650℃，平均株高100cm，茎秆坚韧抗倒伏，分力强，株型紧凑，平均穴穗数27.2穗，平均穗粒数116.3粒，结实率90%以上，千粒重24.4g，谷粒长形，长宽比为2.2，无芒，颖壳黄色。
B、"白梗一号"水稻是晚熟品种，生育期150天，需活动积温3000℃，平均株高120cm，茎秆脆弱易倒伏，分力弱，株型松散，平均穴穗数20穗，平均穗粒数90粒，结实率80%，千粒重30g，谷粒短圆形，长宽比为1.5，有芒，颖壳绿色。
C、"白梗一号"水稻是中晚熟品种，生育期140天，需活动积温2800℃，平均株高110cm，茎秆坚韧抗倒伏，分力中等，株型紧凑，平均穴穗数25穗，平均穗粒数100粒，结实率85%，千粒重26g，谷粒长形，长宽比为2.0，无芒，颖壳红色。
D、"白梗一号"水稻是早熟品种，生育期120天，需活动积温2500℃，平均株高90cm，茎秆脆弱易倒伏，分力强，株型松散，平均穴穗数30穗，平均穗粒数130粒，结实率95%，千粒重22g，谷粒短圆形，长宽比为1.8，有芒，颖壳白色。]
{正确答案：A。}
</example>

Example Dialogue

**User:**
请基于给定学术论文，生成*N*个问答对。

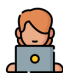

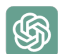

**Example:**
问题1："甬优17"水稻的米质特性如何？
回答1：[选项：A、整精米率71.3%，长宽比2.3，垩白粒率14.0%，垩白度2.0%，透明度2级，胶稠度70mm，直链淀粉含量14.4%，蛋白质含量8.2%。
B、整精米率65.0%，长宽比2.0，垩白粒率20.0%，垩白度3.0%，透明度1级，胶稠度60mm，直链淀粉含量12.0%，蛋白质含量7.0%。
C、整精米率75.0%，长宽比2.5，垩白粒率10.0%，垩白度1.5%，透明度3级，胶稠度80mm，直链淀粉含量16.0%，蛋白质含量9.0%。
D、整精米率68.0%，长宽比2.1，垩白粒率18.0%，垩白度2.5%，透明度2级，胶稠度65mm，直链淀粉含量13.0%，蛋白质含量7.5%。]
{正确答案：A。}
问题2：……
回答2：……

Figure 31: Prompt and few examples for Chinese Multi-Choice Q&A.

Table 14: Performance of LLMs trained under different epochs. The highest value for each model on different tasks is **bolded**. Variation denotes the accuracy difference between Chinese and English.

| Model | Epoch | Size | Overall ↑ | Difficulty | | |
|---|---|---|---|---|---|---|
| | | | | Easy ↑ | Moderate ↑ | Difficult ↑ |
| LLaMA3-Base | N/A | 8B | 0.348 | 0.443 | 0.341 | 0.161 |
| +CROP | 1 | 8B | 0.745 | 0.847 | 0.771 | 0.375 |
| +CROP | 2 | 8B | 0.741 | 0.852 | 0.760 | 0.380 |
| +CROP | 4 | 8B | 0.752 | 0.866 | 0.772 | **0.378** |
| +CQIA+CROP | 1 | 8B | 0.738 | 0.903 | 0.758 | 0.292 |
| +CQIA+CROP | 2 | 8B | 0.742 | 0.902 | 0.772 | 0.271 |
| +CQIA+CROP | 4 | 8B | **0.754** | **0.918** | **0.779** | 0.295 |
| Qwen1.5-Base | N/A | 7B | 0.646 | 0.799 | 0.646 | **0.302** |
| +CROP | 1 | 7B | **0.710** | 0.898 | **0.724** | 0.197 |
| +CROP | 2 | 7B | 0.636 | 0.804 | 0.645 | 0.189 |
| +CROP | 4 | 7B | 0.676 | 0.849 | 0.688 | 0.202 |
| +CQIA+CROP | 1 | 7B | 0.702 | **0.910** | 0.717 | 0.183 |
| +CQIA+CROP | 2 | 7B | 0.670 | 0.875 | 0.677 | 0.181 |
| +CQIA+CROP | 4 | 7B | 0.709 | **0.910** | 0.704 | 0.227 |
| InternLM2-Base | N/A | 7B | 0.368 | 0.445 | 0.381 | 0.148 |
| +CROP | 1 | 7B | 0.666 | 0.841 | 0.651 | 0.294 |
| +CROP | 2 | 7B | 0.622 | 0.818 | 0.600 | 0.230 |
| +CROP | 4 | 7B | 0.748 | 0.945 | 0.761 | 0.212 |
| +CQIA+CROP | 1 | 7B | 0.764 | **0.942** | 0.787 | 0.276 |
| +CQIA+CROP | 2 | 7B | **0.809** | 0.909 | **0.855** | **0.414** |
| +CQIA+CROP | 4 | 7B | 0.768 | 0.939 | 0.794 | 0.285 |

## L More Experiments

To explore how much improvement is credited to CROP, we conduct additional experiments on Table 3, shown in Table 14. We show the performance trend of the model with the increase in the number of iterations when trained on our CROP dataset. It can be observed that in most cases, CROP dominates the performance. However, exceptions occur in some epochs, where the reasons are analyzed:

- Continual Pretraining Phase: All base versions of open-source models in the experiments are only fine-tuned using CROP and CQIA without undergoing continual pretraining. Yet continual pretraining is crucial for language models to effectively learn and retain domain-specific knowledge. Instruction fine-tuning primarily focuses on improving the model's ability to follow instructions rather than acquiring new knowledge.

- Benchmark Specificity: While the CROP dataset is designed for the crop science domain, the CROP benchmark might not fully capture the nuanced improvements brought by the CROP dataset given the significant gap in the amount of raw data. The improvements might be more pronounced in real-world applications or other evaluation metrics to be included in future versions of the benchmark.

